# Temporal Graph Neural Tangent Kernel
# with Graphon-Guaranteed

**Katherine Tieu**[*]
University of Illinois Urbana-Champaign
kt42@illinois.edu

**Dongqi Fu**[*]
Meta AI
dongqifu@meta.com

**Yada Zhu**
IBM Research
yzhu@us.ibm.com

**Hendrik Hamann**
IBM Research
hendrikh@us.ibm.com

**Jingrui He**
University of Illinois Urbana-Champaign
jingrui@illinois.edu

## Abstract

*Graph Neural Tangent Kernel* (GNTK) fuses graph neural networks and graph kernels, simplifies the process of graph representation learning, interprets the training dynamics of graph neural networks, and serves various applications like protein identification, image segmentation, and social network analysis. In practice, graph data carries complex information among entities that inevitably evolves over time, and previous static graph neural tangent kernel methods may be stuck in the sub-optimal solution in terms of both effectiveness and efficiency. As a result, extending the advantage of GNTK to temporal graphs becomes a critical problem. To this end, we propose the temporal graph neural tangent kernel, which not only extends the simplicity and interpretation ability of GNTK to the temporal setting but also leads to rigorous temporal graph classification error bounds. Furthermore, we prove that when the input temporal graph grows over time in the number of nodes, our temporal graph neural tangent kernel will converge in the limit to the *graphon* NTK value, which implies the transferability and robustness of the proposed kernel method, named **Temp**oral **G**raph **N**eural **T**angent **K**ernel with **G**raphon-**G**uaranteed or **Temp-G$^3$NTK**. In addition to the theoretical analysis, we also perform extensive experiments, not only demonstrating the superiority of Temp-G$^3$NTK in the temporal graph classification task, but also showing that Temp-G$^3$NTK can achieve very competitive performance in node-level tasks like node classification compared with various SOTA graph kernel and representation learning baselines. Our code is available at `https://github.com/kthrn22/TempGNTK`

## 1 Introduction

Graphs, as a relational structure, model the complex relationships among entities and have attracted much research attention nowadays. To serve various applications, graph neural networks have been extensively studied for their representation learning ability. On the one hand, graph neural networks usually need to build complex neural architectures with hyperparameters to achieve their powerful expressive ability, which is typically a nonlinear process and hard to interpret [50, 13, 21]. On the other hand, graph kernels enjoy the explicit formula and can be convex, leading to solid theoretical results, although their specific form is often hand-crafted and may not be powerful enough to support complicated application scenarios [23, 34, 12]. Hence, *graph neural tangent kernel* (GNTK) [9] has been proposed to fuse graph neural networks and graph kernels, enjoying the benefits of both

---

[*]Equal Contribution

approaches, i.e., achieving the excellent representation ability while relying on simple computation processes.

However, in the real world, the graph topology and features are inevitably evolving over time, e.g., the user connections and interests in social networks. This temporal evolution brings new challenges to GNTK research as to how the similarity of temporal graphs is measured and how the corresponding kernel matrix is derived. To be more specific, how can we design a temporal graph neural tangent kernel, which not only *has a superior representation ability than temporal graph neural networks [39, 8]* but also *inherits the expression simplicity and analysis rigorousness of graph neural tangent kernels [9, 24]*?

Hence, we propose Temporal Graph Neural Tangent Kernel with Graphon-Guaranteed, or Temp-G$^3$NTK. **First**, we propose the kernel matrix computation formula for temporal graphs with time-evolving structures and time-evolving node features, and the corresponding kernel value can be used for classification tasks with generalization error bounded. This proposed kernel method addresses how to measure the similarity between temporal graphs to achieve the accuracy of graph neural networks but without complex neural computational procedures like gradient descent. **Second**, considering the property of temporal graphs, we also prove that when the temporal graph is growing, i.e., the number of nodes increases over time, our Temp-G$^3$NTK kernel will converge in the limit to the graphon NTK value. This result addresses the challenge of adapting graphon neural network [41] and graphon neural tangent kernel [24] to the temporal setting, and, more importantly, demonstrates that Temp-G$^3$NTK has the excellent potential to transfer to large-scale temporal graph data with robust performance. **Third**, in addition to the theoretical analysis, we also design extensive experiments for not only temporal graph classification but also temporal node classification, illustrating the effectiveness of the proposed Temp-G$^3$NTK compared with various state-of-the-art temporal graph representation learning and graph kernel methods.

## 2 Temporal Graph Modeling

To begin with, we first denote a static undirected graph as $G = (V, E)$, where $V$ and $E$ are sets of vertices and edges, respectively. We also denote the node features of node $v$ ($v \in V$) as $\mathbf{h}_v \in \mathbb{R}^d$, the neighborhood as $\mathcal{N}(v)$, and edge features of an edge $(u, v)$ as $\mathbf{e}_{uv}$.

In order to extend to the temporal setting, researchers usually model the temporal graph $G$ as a *continuous-time dynamic graph* (CTDG) [22], which is mathematically represented as a stream of events, $G = \{(u, v, \mathbf{h}_u(t), \mathbf{h}_v(t), \mathbf{e}_{uv}(t), t)\}_{t=t_0}^{T}$, where an event $(u, v, \mathbf{h}_u(t), \mathbf{h}_v(t), \mathbf{e}_{uv}(t), t)$ indicates that at time $t$, an edge exists between node $u$ and $v$, and $\mathbf{h}_u(t)$, $\mathbf{h}_v(t)$, and $\mathbf{e}_{uv}(t)$ are the features of $u, v$, and $(u, v)$ at time $t$, respectively. To support different computation requirements, a CTDG $G$ can also be transferred into a *discrete-time dynamic graph* (DTDG) [22], which is a collection of snapshots $G^{(t)}$. To be specific, a snapshot of $G$ at any time $t \geq t_0$, is denoted as $G^{(t)}$, which can be obtained by sequentially updating the initial state of $G^{(t_0)}$ with the event stream, i.e., $G^{(t)} = \{(u, v, \mathbf{h}_u(\bar{t}), \mathbf{h}_v(\bar{t}), \mathbf{e}_{uv}(\bar{t}), \bar{t})\}_{\bar{t}=t_0}^{t}$ given $(t_0 \leq \bar{t} \leq t)$, and temporal graph equals to the last time snapshot, i.e., $G = G^{(T)}$ given the last timestamp is denoted by $T$.

Let $V^{(t)}, E^{(t)}$ be the sets of vertices and edges of $G^{(t)}$. We denote the temporal neighborhood of node $v$ at time $t$ as $\mathcal{N}^{(t)}(v) = \{(u, \bar{t}) : ((u, v, \mathbf{h}_u(\bar{t}), \mathbf{h}_v(\bar{t}), \mathbf{e}_{uv}\bar{t}) \in G^{(t)}\}$, i.e., a set of nodes $u$ that are involved in an event with $v$ at any time $\bar{t}$ ($t_0 \leq \bar{t} \leq t$). Note that, in the rest of the paper, we use $G$ to denote an entire temporal graph, and $G^{(t)}$ as a snapshot. For simplicity, we denote $t_0 = 0$.

## 3 Preliminaries of Temp-G$^3$NTK

### 3.1 Temporal Graph Representation Learning

Graph Neural Networks (GNN) are a family of neural architectures for graph representation learning. In general, most GNNs leverage a message-passing framework to compute a node $v$'s representation $\mathbf{h}_v^{(l)}$ at the $l^{th}$ layer. In the static setting, $\mathbf{h}_v^{(l)}$ can be obtained by applying a neighborhood aggregation operator on $\mathbf{h}_u^{(l-1)}$, $\forall u \in \mathcal{N}(v)$, then transforming the aggregated neighborhood information. The graph-level representation can be obtained by applying a pooling function on representations of all nodes, e.g., the summation of all node representations.

To derive Temp-G$^3$NTK, our first step is to compute node representations at time $t$ and then apply a pooling function over all nodes to obtain the graph-level (or snapshot-level) representation. Specifically, we obtain the node representation of $v$ at time $t$, $\mathbf{h}_v(t)$, by aggregating information from its temporal neighborhood $\mathcal{N}^{(t)}(v)$ as

$$\mathbf{h}_v(t) = c \cdot \sum_{(u,\bar{t}) \in \mathcal{N}^{(t)}(v)} [\mathbf{t}_{enc}(t - \bar{t}) || \mathbf{e}_{uv}(\bar{t}) || \mathbf{x}_u(\bar{t})] \tag{1}$$

where $\mathbf{x}_u(t)$ is the node feature of $u$ at time $t$, $\mathbf{t}_{enc} : \mathbb{R} \to \mathbb{R}^{d_t}$ is the time encoding function that can be instantiated [8] as $\mathbf{t}_{enc}(\Delta t) = \cos(\Delta t \mathbf{w})$, and $\mathbf{w} \in \mathbb{R}^{d_t}$ with its $i^{\text{th}}$ entry $[\mathbf{w}]_i = \alpha^{-(i-1)/\beta}$ and $d_t$ is the dimension of the time representation vector. Operation $[\cdot||\cdot]$ denotes the concatenation. $c$ is the scaling factor, and if we set $c = 1$, then Eq. 1 is simply the sum neighborhood aggregation; and if $c = |\frac{1}{|\mathcal{N}^{(t)}(v)|}|$, then Eq. 1 would be the average neighborhood aggregation. Note that if the edge features do not exist then we simply set $\mathbf{e}_{uv}(t) = 0$. Similarly, if node features are not available then we let $\mathbf{x}_u(t) = 0$.

After aggregating information from node $v$'s neighborhood at time $t$ as Eq. 1, we can input $\mathbf{h}_v(t)$ into $L$ layers of Multilayer Perceptrons (MLPs), where the representation of node $v$ after the $l^{th}$ MLP projection is as

$$\mathbf{h}_v^{(l)}(t) = \sqrt{\frac{2}{d_l}} \sigma\big(\mathbf{W}^{(l)} \mathbf{h}_v^{(l-1)}(t)\big) \tag{2}$$

where $d_l$ is the output dimension of the $l^{\text{th}}$ MLP layer, $\sigma$ is a non-linear activation function that can be instantiated as the Rectified Linear Units (ReLU) function.

Furthermore, we denote the graph-level representation of $G^{(t)}$ as $\mathbf{h}_G(t) = \sum_{v \in V^{(t)}} \mathbf{h}_v^{(L)}(t)$.

## 3.2 Graph Neural Tangent Kernel

Next, we provide some background on the infinite-width limit of a fully connected deep neural network $f_{nn}$ and derive the definition of NTK and its properties on graphs.

Consider the following settings: given a training dataset of $n$ samples $\{(\mathbf{x}_i, y_i)\}_{i=1}^n$, where $\mathbf{x}_i \in \mathbb{R}^d$ and its label denoted by $y_i \in \mathbb{R}$. Let $f_{nn}(\mathbf{x}, \theta)$ be the output of a fully-connected neural network, with parameters $\theta \in \mathbb{R}^p$ and $\mathbf{x}$ as the input. We train $f_{nn}$ by minimizing the squared loss over the training dataset.

$$\ell(\theta) = \frac{1}{2} \sum_{i=1}^n (f_{nn}(\mathbf{x}_i, \theta) - y_i)^2 \tag{3}$$

Let $\mathbf{X} \in \mathbb{R}^{n \times d}$ (where $[\mathbf{X}]_i = \mathbf{x}_i$) and $\mathbf{y} \in \mathbb{R}^n$ (where $[\mathbf{y}_i] = y_i$), such that $f_{nn}(\mathbf{X}, \theta)$ would be the prediction of $f_{nn}$, with parameters $\theta$, over all $\mathbf{x}_i$ of the training set.

Suppose that $\ell$ is minimized by gradient descent, so the output $f_{nn}(\mathbf{X}, \theta)$ evolves with respect to the training time $\tau$ as follows [1].

$$\frac{d\, f_{nn}(\mathbf{X}, \theta(\tau))}{d\tau} = -\mathbf{H}(\tau) \cdot (f_{nn}(\mathbf{X}, \theta(\tau)) - \mathbf{y}) \tag{4}$$

where $\theta(\tau)$ is the parameters $\theta$ being updated at training time $\tau$ based on gradient descent, and $\mathbf{H}(\tau)$ is a $n \times n$ positive definite matrix with its $(i,j)$-th entry as follows

$$\left\langle \frac{\partial f_{nn}(\mathbf{x}_i, \theta(\tau))}{\partial \theta}, \frac{\partial f_{nn}(\mathbf{x}_j, \theta(\tau))}{\partial \theta} \right\rangle \tag{5}$$

Existing works on over-parameterized neural networks [1, 2], [10, 11], and [19] have proven that for infinite-width neural networks, the matrix $\mathbf{H}(\tau)$ remains constant during training, and under random initialization of parameters, the matrix $\mathbf{H}(0)$ converges in probability to a certain deterministic kernel matrix $\mathbf{H}^*$, which is named as Neural Tangent Kernel [19]. Moreover, as proven in [1, 2], the prediction of a fully-trained sufficiently wide ReLU neural network is equivalent to the kernel regression predictor with the kernel matrix $\mathbf{H}^*$.

For the temporal setting, similar to the NTK and the infinite-width neural networks, let $f_{temp}$ denote the aforementioned temporal GNN in Section 3.1, and $f_{temp}(G^{(t)}, \mathbf{W})$ be the output of $f_{temp}$ with the input $G^{(t)}$ and parameters $\mathbf{W}$. Given two temporal graphs $G$ and $G'$, at time $t$, the NTK value corresponds to infinite-width $f_{temp}$, i.e., in the limit that $d_l \to \infty$, where $d_l$ is the output dimension stated in Eq. 2, $l \in [L]$, such that

$$K(G^{(t)}, G'^{(t)}) = \mathbb{E}_{\mathbf{W} \sim \mathcal{N}(0,1)} \left\langle \frac{\partial f(G^{(t)}, \mathbf{W})}{\partial \mathbf{W}}, \frac{\partial f(G'^{(t)}, \mathbf{W})}{\partial \mathbf{W}} \right\rangle \tag{6}$$

Then, in the rest of this paper, we can refer to this value $K(G^{(t)}, G'^{(t)})$ as the Temp-G$^3$NTK value of $G$ and $G'$ at time $t$. In the next section, we are ready to introduce how to compute the defined kernel as Eq. 6 without training neural networks.

# 4 Proposed Temp-G$^3$NTK

Given two temporal graphs $G$ and $G'$, we propose to compute the Temp-G$^3$NTK value at time $t$, i.e., $K(G^{(t)}, G'^{(t)})$, with $L$ BLOCK operations [2]. We discuss the detailed computation procedure of Temp-G$^3$NTK here and leave the theoretical derivation in Section 5.

In general, similar to [1, 9], we first recursively compute the node pairwise covariance matrix $\mathbf{\Sigma}^{(l)}$, its derivative $\dot{\mathbf{\Sigma}}^{(l)}$, and the node-pairwise kernel matrix $\mathbf{\Theta}^{(l)}$ that correspond to the $l^{\text{th}}$ BLOCK transformation. Finally, the Temp-G$^3$NTK value is obtained by the summation of all of the entries in the kernel matrix of the last BLOCK transformation, i.e., $\mathbf{\Theta}^{(L)}$.

To begin with, we initialize the node pairwise covariance matrix $\mathbf{\Sigma}$ and the kernel matrix $\mathbf{\Sigma}$ at entries $u, u'$ ($u \in V^{(t)}, u' \in V'^{(t)}$) as the inner product of node representations of $u, u'$ at time $t$, respectively,

$$\mathbf{\Theta}^{(0)}(G^{(t)}, G'^{(t)})_{uu'} = \mathbf{\Sigma}^{(0)}(G^{(t)}, G'^{(t)})_{uu'} = \mathbf{h}_u(t)^T \mathbf{h}'_u(t) \tag{7}$$

where $\mathbf{h}_u(t)$ and $\mathbf{h}'_u(t)$ are computed by Eq. 1.

Next, we need to compute $\mathbf{\Sigma}^{(l)}$ and $\mathbf{\Theta}^{(l)}$ that correspond to the $l^{\text{th}}$ BLOCK operator with ReLU activation function $\sigma$. As $\sigma(x) = \max(0, x)$, the derivative of $\sigma$ is $\dot{\sigma}(x) = \mathbb{1}[x \geq 0]$, where $\mathbb{1}$ is the indicator vector[3]. For $l(1 \leq l \leq L)$, we first define an intermediate covariance matrix as

$$\mathbf{\Lambda}^{(l)}(G^{(t)}, G'^{(t)})_{uu'} = \begin{pmatrix} \mathbf{\Sigma}^{(l-1)}(G^{(t)}, G'^{(t)})_{uu'} & \mathbf{\Sigma}^{(l-1)}(G^{(t)}, G^{(t)})_{uu} \\ \mathbf{\Sigma}^{(l-1)}(G'^{(t)}, G^{(t)})_{u'u} & \mathbf{\Sigma}^{(l-1)}(G'^{(t)}, G'^{(t)})_{u'u'} \end{pmatrix} \tag{8}$$

and $\mathbf{\Lambda}^{(l)}(G^{(t)}, G'^{(t)})_{uu'} \in \mathbb{R}^{2 \times 2}$.

As the covariance matrix $\mathbf{\Sigma}^{(l)}$ represents the i.i.d centered Gaussian Processes of $h_u(t)$ and $h'_u(t)$ after transformed by $l$ BLOCK operations, we can compute $\mathbf{\Sigma}^{(l)}$ and $\dot{\mathbf{\Sigma}}^{(l)}$ based on the aforementioned intermediate covariance matrix as

$$\mathbf{\Sigma}^{(l)}(G^{(t)}, G'^{(t)})_{uu'} = \mathbb{E}_{(a,b) \sim \mathcal{N}(0, \mathbf{\Lambda}^{(l)}(G^{(t)}, G'^{(t)})_{uu'})}[\sigma(a) \cdot \sigma(b)]$$
$$= \frac{\pi - \arccos(\mathbf{\Sigma}^{(l-1)}(G^{(t)}, G'^{(t)})_{uu'})}{2\pi} + \frac{\sqrt{1 - (\mathbf{\Sigma}^{(l-1)}(G^{(t)}, G'^{(t)})_{uu'})^2}}{2\pi} \tag{9}$$

with

$$\dot{\mathbf{\Sigma}}^{(l)}(G^{(t)}, G'^{(t)})_{uu'} = \mathbb{E}_{(a,b) \sim \mathcal{N}(0, \mathbf{\Lambda}^{(l)}(G^{(t)}, G'^{(t)})_{uu'})}[\dot{\sigma}(a) \cdot \dot{\sigma}(b)]$$
$$= \frac{\pi - \arccos(\mathbf{\Sigma}^{(l-1)}(G^{(t)}, G'^{(t)})_{uu'})}{2\pi} \tag{10}$$

Eq. 9 and Eq. 10 hold due to the closed form of the kernel function with ReLU activation $g(x, y) = \mathbb{E}_w[\sigma'(w^T x)\sigma'(w^T y)] = \left(\frac{1}{2} - \frac{\arccos x^T y}{2\pi}\right)$.

Then, the $l^{\text{th}}$ kernel matrix, $\boldsymbol{\Theta}^{(l)}$, is obtained as

$$\boldsymbol{\Theta}^{(l)}(G^{(t)}, G'^{(t)})_{uu'} = \boldsymbol{\Theta}^{(l-1)}(G^{(t)}, G'^{(t)})_{uu'} \cdot \dot{\boldsymbol{\Sigma}}^{(l)}(G^{(t)}, G'^{(t)})_{uu'} + \boldsymbol{\Sigma}^{(l)}(G^{(t)}, G'^{(t)})_{uu'} \quad (11)$$

Finally, the Temp-G$^3$NTK value of $G, G'$ at time $t$ is

$$K(G^{(t)}, G'^{(t)}) = \sum_{v \in V^{(t)}} \sum_{v' \in V'^{(t)}} \boldsymbol{\Theta}^{(L)}(G^{(t)}, G'^{(t)})_{vv'} \quad (12)$$

We perform summation over all entries since in our proposed neural architecture in Section 3.1, we can obtain the graph embedding by applying a pooling function, e.g., sum pooling, on node-level representations.

The pseudo-code of computing the Temp-G$^3$NTK kernel as above is shown in Appendix A.

## 5 Theoretical Analysis of Temp-G$^3$NTK

### 5.1 Kernel Properties of Temp-G$^3$NTK

To begin with, we first show that our proposed kernal function Temp-G$^3$NTK satisfies symmetric and semi-definite below, and the full proof can be found in Appendix B

**Theorem 5.1.** *Temp-G$^3$NTK is symmetric.*

**Theorem 5.2.** *Temp-G$^3$NTK is positive semi-definite.*

### 5.2 Generalization Bound of Temp-G$^3$NTK

We first state how to utilize Temp-G$^3$NTK as a kernel regression predictor for the temporal graph classification task; then, we establish a data-dependent generalization error bound of the function class of kernel regression predictors that are associated with Temp-G$^3$NTK.

To be more specific, we can instantiate the problem of temporal graph classification, where, given an i.i.d training set of $n$ temporal graphs $\{G_1, G_2, \ldots, G_n\}$ and their labels $\{y_1, y_2, \ldots, y_n\}$, our goal is to predict the label [4] of a testing temporal graph $G_{test}$. Then, the prediction of $G_{test}^{(t)}$ at any time $t$ by a kernel regression predictor $f_{kernel}$ associated with Temp-G$^3$NTK kernel $K(\cdot, \cdot)$ is expressed as follows,

$$f_{kernel}(G_{test}^{(t)}) = \left[K(G_{test}^{(t)}, G_1^{(t)}), \ldots, K(G_{test}^{(t)}, G_n^{(t)})\right] [\mathbf{K}_{train}^{(t)}]^{-1}\mathbf{y} \quad (13)$$

where $\mathbf{K}_{train}^{(t)}$ is a positive definite $n \times n$ kernel matrix, whose $(i, j)$-th entry is the Temp-G$^3$NTK value of $G_i^{(t)}, G_j^{(t)}$, i.e., $[\mathbf{K}_{train}^{(t)}]_{i,j} = K(G_i^{(t)}, G_j^{(t)})$ and $\mathbf{y} \in \mathbb{R}^n$ is the label space of temporal graphs, whose $i^{\text{th}}$ entry is $[\mathbf{y}]_i = y_i$.

Then, we consider any loss function $\ell : \mathbb{R} \times \mathbb{R} \to [0, 1]$ that is $\alpha-$Lipschitz. We define the generalization error of the predictor $f_{kernel}$ in Eq. 13 at time $t$ that acts on a temporal graph $G$ labeled by $y$ as

$$\mathbb{E}[\ell(f_{kernel}(G^{(t)})y)|\{G^{(1)}, \ldots, G^{(t-1)}\}] - \ell(f_{kernel}(G^{(t)}), y) \quad (14)$$

where the expectation is taken over all $G^{(t)}$ drawn from the stochastic process $\mathbb{P}_t(.|G^{(1)}, \ldots, G^{(t-1)})$ conditioned on all previous snapshots before time $t$ of temporal graph $G$. The following theorem establishes the generalization error bound on all snapshot $G^{(t)}$ of $G$.

**Theorem 5.3.** *Given $n$ i.i.d training samples and their labels $\{G_i, y_i\}_{i=1}^n$ and $G_i$ has $t$ timestamps, let $\mathbf{K}_{train}^{(t)} \in \mathbb{R}^{n \times n}$ be the kernel matrix of pairwise Temp-G$^3$NTK values between graphs of the training set at time $t$ and $f_{kernel}$ be the kernel regression predictor based on the training set and*

$\mathbf{K}_{train}^{(t)}$. *Consider any loss function $\ell : \mathbb{R} \times \mathbb{R} \rightarrow [0,1]$ that is $\alpha-$Lipschitz, the generalization error of the $f_{kernel}$ predictor can be upper bounded as*

$$\sup_{\ell \in \mathcal{L}} \left[ \frac{1}{T} \sum_{t=1}^{T} \mathbb{E}[\ell(f_{kernel}(G^{(t)}), y) | \{G^{(1)}, \ldots, G^{(t-1)}\}] - \ell(f_{kernel}(G^{(t)}), y) \right]$$

$$\leq \mathcal{O}\left( \sup_{t} \mathbf{y}^T [\mathbf{K}_{train}^{(t)}]^{-1} \mathbf{y} \cdot \text{tr}(\mathbf{K}_{train}^{(t)}) \right) \tag{15}$$

*where $\mathcal{L}$ is the class containing all $\alpha-$Lipschitz functions, the expectation is taken over all $G^{(t)}$ that is drawn from $\mathbb{P}_t(\cdot | G^{(1)}, \ldots, G^{(t-1)})$.*

In brief, inspired by existing works on generalization bounds for kernel classes [4], we first bound our generalization error by the Sequential Rademacher Complexity [38, 26] of $\mathcal{F}$ (i.e., the function class containing kernel such as $f_{kernel}$), and then bound this complexity measure by $\mathcal{O}(\sup_t \mathbf{y}^T [\mathbf{K}_{train}^{(t)}]^{-1} \mathbf{y} \cdot \text{tr}(\mathbf{K}_{train}^{(t)}))$, where $\sup_t \mathbf{y}^T [\mathbf{K}_{train}^{(t)}]^{-1} \mathbf{y} \cdot \text{tr}(\mathbf{K}_{train}^{(t)}))$ gives maximum value of $\mathbf{y}^T [\mathbf{K}_{train}^{(t)}]^{-1} \mathbf{y} \cdot \text{tr}(\mathbf{K}_{train}^{(t)}))$ over all timestamps of the training temporal graphs. The classification to the temporal graph $G$ is the max-aggregation of $f_{kernel}(G^{(t)})$. The full proof is in Appendix C.

## 5.3 Convergence of Temp-G$^3$NTK

In this part, we investigate our Temp-G$^3$NTK value on two growing temporal graphs, $G$ and $G'$. "*Growing*" means the number of nodes in $G$ and $G'$ would increase with time, and the following theorem shows that the proposed Temp-G$^3$NTK enjoys a rigorous convergence. To verify this, we first adopt the definition of *Graphon NTK* on a single growing graph [24] and then extend the concept to different and temporal graphs to establish the convergence of Temp-G$^3$NTK value of $G, G'$ as follows. The full proof is provided in Appendix D.

**Theorem 5.4.** *Given two growing temporal graphs $G$ and $G'$ and two graphons $W$ and $W'$, suppose snapshots of $G$ (i.e., $G^{(t)}$) converge to $W$ and snapshots of $G'$ (i.e., $G'^{(t)}$) converge to $W'$, as $t \rightarrow \infty$. Then, the graphon neural tangent kernel induced by Temp-G$^3$NTK of $G, G'$ at time $t$, i.e., $K_W(W^{(t)}, W'^{(t)})$, converges in the limit of the operator norm to the graphon neural tangent kernel of $W$ and $W'$, i.e., $K_W(W, W')$, as follows:*

$$\lim_{t \rightarrow \infty} ||K_W(W^{(t)}, W'^{(t)}) - K_W(W, W')|| \rightarrow 0 \tag{16}$$

*where $K_W$ denotes the graphon NTK value.*

This theorem addresses the convergence limitations of previous work [24] in terms of different temporal graphs. In other words, besides temporal dependencies between snapshots of different evolving graphs, the work [24] only establishes a limit object for different stages of a single growing graph. An empirical visualization can be seen in Figure 1, and the detailed comparison and illustration are delivered in the Appendix D.3.

## 5.4 Time Complexity of Temp-G$^3$NTK

Here, the following table shows the time complexity comparison between our Temp-G$^3$NTK with other graph kernel and graph representation learning methods for measuring $n$ pairs of temporal graphs at a certain timestamp $t$.

In the above table, we first need to declare some mathematical notation as follows. $|V|, |E|$ denote the maximum size of the vertex set and edge set among $n$ given graphs. Then, for the time complexity of WL-Subtree [42], Graph2Vec [33], and GL2Vec [6], $h$ denotes the number of iterations in WL-Subtree algorithms; for Graph2Vec [33] and GL2Vec [6], $D$ represents the maximum degree of the rooted subgraphs that are used to compute graph embeddings; and for the time complexity of NetLSD [45], $k$ denotes the number of eigenvalues (obtained from the graph Laplacian matrix) used to compute the graph embeddings; for TGN [39], $L_{hop}$ denotes the number of neighbor hops that a node can aggregate information from; for our Temp-G$^3$NTK, based on Section 4 and Appendix A, $L$

Table 1: Total Runtime Complexity of Computing Similarity for $n$ Pairs of Graphs at Timestamp $t$.

| METHOD | RUNTIME COMPLEXITY |
|--------|--------------------|
| WL-SUBTREE [42] | $\mathcal{O}(nh|E| + n^2h|V|)$ |
| SHORTEST PATH [5] | $\mathcal{O}(n^2|V|^4)$ |
| RANDOM WALK [46] | $\mathcal{O}(n^2|V|^3)$ |
| GRAPH2VEC [33] | $\mathcal{O}(n^2|V|Dh|E|) \cdot \mathcal{B}$ |
| NETLSD [45] | $\mathcal{O}(n^2(k|E| + k^2|V|)) \cdot \mathcal{B}$ |
| GL2VEC [6] | $\mathcal{O}(n|V|^2 + n^2|V|Dh|E|) \cdot \mathcal{B}$ |
| GRAPHMIXER [8] | $\mathcal{O}(n^2 + n|V|K) \cdot \mathcal{B}$ |
| TGN [39] | $\mathcal{O}(n^2 + n(|V| + |E|)L_{hop}) \cdot \mathcal{B}$ |
| TEMP-G$^3$NTK (OURS) | $\mathcal{O}(n^2L|V|^2 + n|E|)$ |

represents the number of BLOCK operations; and $\mathcal{B}$ denotes the number of training epochs for all neural representation learning algorithms.

Notably, our method Temp-G$^3$NTK falls into the category of graph kernels, and its computational complexity is cheaper than [5, 46] . Also, compared with graph neural representation methods [6, 33, 45, 39, 8], the computation iteration of Temp-G$^3$NTK does not rely on neural computation like gradient descent and backpropagation, such that the empirical execution time of our method is still faster. Moreover, we further demonstrate our Temp-G$^3$NTK's efficiency by providing empirical runtime comparison in Table 3, and the detailed empirical effectiveness comparison of these methods is shown in the next section.

# 6 Experiments

In this section, we demonstrate the performance of Temp-G$^3$NTK by crucial tasks of temporal graph learning. More extra experiments about *ablation study*, *parameter analysis*, and *robustness* can be referred to Appendix F.

## 6.1 Graph-Level Experiments

**Datasets**. Targeting temporal graph classification, we conduct experiments on one of the most advanced temporal graph benchmarks that have graph-level labels, i.e., TUDataset [5] [32], the four datasets are INFECTIOUS, DBLP, FACEBOOK, and TUMBLR, the detailed dataset statistics can also be found in Appendix G.1. Additionally, we also leveraged the more large-scale temporal datasets REDDIT, WIKIPEDIA, LASTFM, and MOOC from [25][6]. Those datasets are large but do not have graph-level labels, so we use them to demonstrate the scalability of Temp-G$^3$NTK on temporal graph similarity measurement. The detailed dataset statistics can be found in Appendix G.1, and corresponding experimental results can be found in Appendix F.3. Below, we focus on introducing temporal graph classification experiments and findings.

**Problem Setting**. For each dataset above, we evaluate the temporal graph classification accuracy by conducting 5-fold cross-validation and then report the mean and standard deviation of test accuracy. To be specific, given a dataset of $n$ temporal graphs $\{G_1, G_2, ..., G_n\}$ and their labels $\{y_1, y_2, ..., y_n\}$, and in all four datasets, label $y_i$ of the temporal graph $G_i$ is already time-aware, which means the value does not change with respect to time. Also, edge features are not provided in these four datasets, and we apply the Temp-G$^3$NTK formula with plain time encoding as stated in Eq. 1.

**Baselines**. We compare Temp-G$^3$NTK with a range of graph classification algorithms: (1) Graph Kernels, including WL-Subtree Kernel [42], Random Walk Kernel [46], and Shortest-Path Kernel [5]; (2) Graph Representation Learning methods, including Graph2Vec [33], NetLSD [45], GL2Vec [6]; and (3) Temporal Graph Representation Learning algorithms, including TGN [39], GraphMixer [8], EvolveGCN [35]. Details about the implementation and parameters of each algorithm are deferred to Appendix G.

Table 2: Comparison of Temporal Graph Classification Accuracy.

| METHOD | INFECTIOUS | DBLP | FACEBOOK | TUMBLR |
|---|---|---|---|---|
| WL-SUBTREE [42] | $0.600 \pm 0.044$ | $0.520 \pm 0.068$ | $0.650 \pm 0.075$ | $0.570 \pm 0.121$ |
| SHORTEST PATH [5] | $0.670 \pm 0.075$ | $0.560 \pm 0.049$ | $0.560 \pm 0.086$ | $0.580 \pm 0.143$ |
| RANDOM WALK [46] | $0.670 \pm 0.073$ | $0.530 \pm 0.058$ | $0.590 \pm 0.093$ | $0.580 \pm 0.112$ |
| GRAPH2VEC [33] | $0.565 \pm 0.081$ | $0.539 \pm 0.031$ | $0.538 \pm 0.028$ | $0.547 \pm 0.071$ |
| NETLSD [45] | $0.625 \pm 0.061$ | $0.558 \pm 0.035$ | $0.535 \pm 0.011$ | $0.552 \pm 0.046$ |
| GL2VEC [6] | $0.545 \pm 0.051$ | $0.562 \pm 0.030$ | $0.538 \pm 0.031$ | $0.558 \pm 0.080$ |
| GRAPHMIXER [8] | $0.500 \pm 0.000$ | $0.563 \pm 0.011$ | $0.561 \pm 0.023$ | $0.509 \pm 0.508$ |
| TGN [39] | $0.520 \pm 0.019$ | $0.580 \pm 0.003$ | $0.559 \pm 0.018$ | $0.517 \pm 0.025$ |
| EVOLVEGCN [35] | $0.521 \pm 0.093$ | $0.400 \pm 0.089$ | $0.516 \pm 0.075$ | $0.395 \pm 0.089$ |
| TEMP-G$^3$NTK (OURS) | $\mathbf{0.740 \pm 0.058}$ | $\mathbf{0.600 \pm 0.063}$ | $\mathbf{0.700 \pm 0.138}$ | $\mathbf{0.630 \pm 0.068}$ |

**Results**. The graph classification results are shown in Table 2, and the best test accuracy is highlighted in bold. Our method, Temp-G$^3$NTK, outperforms the other methods on all four datasets. In particular, the most notable gap between Temp-G$^3$NTK and the other methods lies in the FACEBOOK dataset, where Temp-G$^3$NTK gains 70% accuracy. In addition, as the label for each graph remains unchanged, we evaluate the performance of baseline algorithms on different timestamps until the end of the temporal graph, and report their highest accuracy score in Table 2.

We also provide a better illustration of how baseline algorithms perform at different timestamps of the INFECTIOUS and FACEBOOK datasets through Figure 1. For Figure 1, as stated in our problem setting, each temporal graph is associated with a label, and the label is fixed across timestamps. Therefore, we expect our method to perform well, i.e., achieve a competitive accuracy score across all timestamps. As illustrated, Figure 1 shows that Temp-G$^3$NTK performs robustly across all timestamps and achieves the highest accuracy at most times, which also recalls the convergence ability of Temp-G$^3$NTK as proved in Section 5.3.

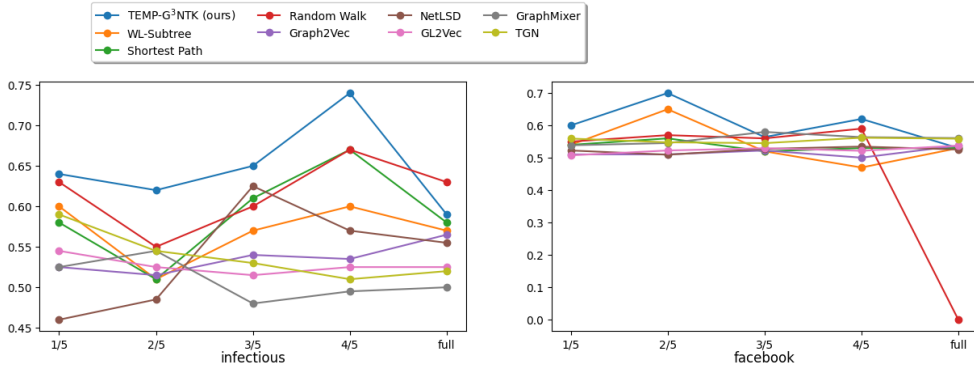

Figure 1: Comparison of *test accuracy* with respect to *different stages of temporal graphs* from the INFECTIOUS and FACEBOOK datasets. The $y$-axis in each plot is the accuracy, and the $x$-axis represents what percentage of timestamps have been taken into account. For example, at $x = 1/5$, the accuracy is obtained by performing classification on the first $1/5$ timestamps of each graph.

Further, in Table 3, we present the runtime comparison for four datasets, INFECTIOUS, DBLP, FACEBOOK, and TUMBLR. Overall, the empirical running time aligns with our theoretical analysis of time complexity in Table 1. That is, our method belongs to the graph kernel category, where the node-wise comparison is usually inevitable, and our time complexity is lower. Compared to the neural baselines, since our method does not rely on complex neural training like gradient descent and backpropagation, our method is still efficient.

Given our method achieved the best classification accuracy, as shown in Table 2, according to the corresponding running time reported in Table 3, our method is (1) more than 10x - 20x faster then complex temporal graph neural network methods like GraphMixer [8] and TGN [39]; (2) similarly efficient as simple kernel methods like WL-Subtree [42] and Shortest Path [5] and embedding

Table 3: Runtime of Baselines for Each Dataset in Seconds

| METHOD | INFECTIOUS | DBLP | FACEBOOK | TUMBLR |
|---|---|---|---|---|
| WL-SUBTREE | 16.04 | 10.93 | 13.88 | 9.80 |
| SHORTEST PATH | 20.36 | 16.13 | 32.29 | 16.29 |
| RANDOM WALK | 489.65 | 566.64 | 5380.26 | 972.929 |
| GRAPH2VEC | 3.43 | 3.45 | 3.87 | 3.42 |
| NETLSD | 14.82 | 15.20 | 33.90 | 15.06 |
| GL2VEC | 17.75 | 14.36 | 23.95 | 13.77 |
| GRAPHMIXER | 217.70 | 537.22 | 720.16 | 219.29 |
| TGN | 254.59 | 873.03 | 1101.66 | 394.34 |
| TEMP-G$^3$NTK (OURS) | 23.04 | 21.00 | 25.86 | 21.04 |

methods like NetLSD [45] and GL2Vec [6]; and only Graph2Vec [33] is running faster than our method, but our performance is roughly 1.4x better.

## 6.2 Node-Level Experiments

In this section, we evaluate the performance of Temp-G$^3$NTK for the temporal node property prediction task. Specifically, we leverage the final node-pairwise kernel matrix computed by Eq. 11, i.e., $\Theta^{(L)}$, and obtain node predictions by performing kernel regression with $\Theta^{(L)}$.

**Datasets**. We demonstrate Temp-G$^3$NTK's capability of performing dynamic node prediction on the tgbn-trade dataset from the Temporal Graph Learning Benchmark (TGB) [17], and the details of TGB can be found at this link [7]. The training, validation, and test sets of tgbn-trade are defined in the TGB package with $70\%/15\%/15\%$ chronological splits. To assess the performance of a method on tgbn-trade, we use the normalized discounted cumulative gain (NDCG) metric that is assigned to tgbn-trade in the TGB package.

**Problem Setting**. Given a temporal graph with *node labels that change with respect to time*, the Node Property Prediction task requires the prediction of labels of some nodes at a certain time $t$, given that our predictor can leverage all information about the temporal graphs from the initial timestamps up to some certain timestamp $\bar{t}$ with $\bar{t} < t$. To be specific, the predictor for node labels by using Temp-G$^3$NTK at time $t$ would be:

$$\Theta^{(L)}(G^{(t)}, G^{(\bar{t})})[\Theta^{(L)}(G^{(\bar{t})}, G^{(\bar{t})})]^{-1}\mathbf{y}(t) \tag{17}$$

where $\mathbf{y}(t) \in \mathbb{R}^{n \times d_{label}}$ is a vector whose $i^{\text{th}}$ entry is the label of node $i$ at time $t$, and $d_{label}$ is the dimension of node labels.

Through the lens of kernel regression, $[\Theta^{(L)}(G^{(\bar{t})}, G^{(\bar{t})})]$ acts as the gram matrix, similar to the role of $\mathbf{K}_{train}$ in Eq. 13, and $\Theta^{(L)}(G^{(t)}, G^{(\bar{t})})$ acts as the kernel values between the test and training samples. In order to effectively utilize Temp-G$^3$NTK for node property prediction, we perform kernel regression with C-SVM and employ $\Theta^{(L)}$ as the pre-computed kernel. The regularization parameter, $C$, of our SVM predictor is searched over 120 values evenly sampled from the interval $[10^{-2}, 10^4]$ in log scale. The number of BLOCK operations, $L$, is searched over $\{1, 2, 3\}$, and we obtain the best NDCG score with $L = 1$.

**Baselines**. We compare Temp-G$^3$NTK with deep learning algorithms on the tgbn-trade's leaderboard, which include TGN [39], DyRep [44], and DyGFormer [49]. TGN [39] is discussed in the temporal graph classification task. DyRep [44] is a deep temporal point process model, which is parameterized by a temporal-attentive representation network encoding time evolving structural information into node representations. DyGFormer [49] is a Transformer-based architecture for dynamic graph learning, which learns from nodes' historical first-hop interactions by the neighbor co-occurrence sampling and patching scheme with the Transformer neural architecture.

With selected temporal graph representation learning baseline methods, we then report the NDCG scores of baseline algorithms based on tgbn-trade's leaderboard.

**Results**. The results for temporal node property prediction on tgbn-trade are shown in Table 4, and the best NDCG score is highlighted in bold. Temp-G$^3$NTK achieves very competitive results, with

Table 4: NDCG Score for Node Property Prediction on the tgbn-trade Dataset.

| METHOD | VALIDATION | TEST |
|---|---|---|
| DYGFORMER [49] | **0.408 ± 0.006** | **0.388 ± 0.006** |
| TGN [39] | 0.395 ± 0.002 | 0.374 ± 0.001 |
| DYREP [44] | 0.394 ± 0.001 | 0.374 ± 0.001 |
| TEMP-G$^3$NTK (OURS) | 0.397 ± 0.039 | 0.380 ± 0.008 |

the test NDCG score of 0.380, outperforming TGN and DyRep and approaching DyGFormer very closely, despite that baselines rely on heavy graph neural architectures like graph neural network or graph transformer. These results show that Temp-G$^3$NTK has the potential to extend to the temporal node property prediction task and capture node-level information.

## 7 Related Work

Graph neural representation learning attracts many research interests and serves for many interesting tasks like recommendation [3, 36, 37], time series forecasting [29, 15], and social network analysis [28, 14, 27]. In which research domain, many efforts have been devoted to develop non-neural computations and temporal settings. **Graph Neural Tangent Kernel**. Graph Neural Tangent Kernel (GNTK) [9] introduces a class of graph kernels that corresponds to infinite-width GNNs with sum neighborhood aggregator. Building upon the foundations of GNTK, a line of works unveil different theoretical aspects of GNTK. For example, [20] improves the computation time of constructing the gram matrix of GNTK; [18] studies the behavior of GNTK that aligns with GNNs with large depth; and most relevant to our theoretical results (Theorem 5.4), [24] combines GNTK with the concept of graphons to derive the Graphon Neural Tangent Kernel. **Graphons, Graphon Neural Network, and Graphon Neural Tangent Kernel**. A graphon is a symmetric, bounded, and measurable function $W : [0,1]^2 \to [0,1]$ that acts as the limit object of dense graph sequences and defines a family of similar graphs. Similarly, Graphon Neural Networks (WNNs) [41] are proven to be the limit object of GNNs that operates on a sequence of graphs as the graph's size grows. Graphon Neural Tangent Kernel (WNTK) [24] defines the NTK that resonates with the infinite-width WNNs and proves that the GNTK converges to the corresponding WNTK as the size of the graph grows. **Temporal Graph Learning**. Most temporal graph learning methods are comprised of complex architectures that leverage the message passing framework, a time encoding function that captures time representation and distinguishes different timestamps. Some works also employ recurrent architecture to capture past information and update the node or edge representation at a current time $t$ based on representations of previous time $\bar{t}$, where $\bar{t} < t$. For example, JODIE [25] employs RNN to update the history representation of $v$ at time $t$. TGAT [47] utilizes the self-attenion mechanism (SAM) to compute the temporal representation of node $v$. TGN [39] employs recurrent architecture to capture the history representation of $\mathbf{x}_v(t)$ (similar to JODIE [25]) and then performs neighborhood aggregation to obtain the temporal node representation of $v$ at time $t$, which is similar to TGAT [47]. GraphMixer [8] first constructs edge representation by aggregating raw edge features and then concatenates them with relative difference time encoding. Then, the temporal node representation is determined by aggregating the aforementioned edge representation, $\mathbf{h}_v(t)$. The node representation is further transformed by MLP and Mixer-MLP layers. For a more comprehensive comparison between Temp-G$^3$NTK and previous recurrent neural network works on Temporal Graph Learning, we refer readers to Appendix E, where we provide detailed illustration of more Temporal Graph Learning methods, DGNN [31], EvolveGCN [35], ROLAND [48], and SSGNN [7].

## 8 Conclusion

In this paper, we study the graph neural tangent kernel within the temporal graph setting and propose a temporal graph neural tangent kernel named Temp-G$^3$NTK, which allows the input graph structure and node features to evolve over time and output the pairwise similarity. The proposed Temp-G$^3$NTK enjoys the computational efficiency, expressive representation ability of temporal graph neural networks, and rigorous error bound. Moreover, the proposed Temp-G$^3$NTK also follows the graphon convergence property. Empirically, we not only test Temp-G$^3$NTK in the temporal graph-level experiments and demonstrate its superior accuracy but also extend it to deal with temporal node-level tasks, where Temp-G$^3$NTK also shows competitive performance.

## Acknowledgements

This work is supported by National Science Foundation under Award No. IIS-2117902, the U.S. Department of Homeland Security under Grant Award Number 17STQAC00001-08-01, MIT-IBM Watson AI Lab, and IBM-Illinois Discovery Accelerator Institute - a new model of an academic-industry partnership designed to increase access to technology education and skill development to spur breakthroughs in emerging areas of technology. The views and conclusions are those of the authors and should not be interpreted as representing the official policies of the funding agencies or the government.

## Footnotes

[2] We follow the name of "BLOCK" in [9], which can be understood as an iterative transformation operation. The new version for temporal graphs is expressed in Eq. 8, Eq. 9, Eq. 10, and Eq. 11.

[3] with the pseudo derivative at 0.

[4]Without loss of generality, we assume snapshot $G^{(t)}$ shares the label with its temporal graph $G$ for clear notation.

[5] https://chrsmrrs.github.io/datasets/docs/datasets/

[6] https://snap.stanford.edu/jodie/

[7] `https://tgb.complexdatalab.com/docs/nodeprop/`

[8]The code for TGN is available at: https://github.com/twitter-research/tgn

[9]The code for GraphMixer is available at: https://github.com/CongWeilin/GraphMixer

## References

[1] Sanjeev Arora, Simon S. Du, Wei Hu, Zhiyuan Li, Ruslan Salakhutdinov, and Ruosong Wang. On exact computation with an infinitely wide neural net. In Hanna M. Wallach, Hugo Larochelle, Alina Beygelzimer, Florence d'Alché-Buc, Emily B. Fox, and Roman Garnett, editors, *Advances in Neural Information Processing Systems 32: Annual Conference on Neural Information Processing Systems 2019, NeurIPS 2019, December 8-14, 2019, Vancouver, BC, Canada*, pages 8139–8148, 2019.

[2] Sanjeev Arora, Simon S. Du, Wei Hu, Zhiyuan Li, and Ruosong Wang. Fine-grained analysis of optimization and generalization for overparameterized two-layer neural networks. In Kamalika Chaudhuri and Ruslan Salakhutdinov, editors, *Proceedings of the 36th International Conference on Machine Learning, ICML 2019, 9-15 June 2019, Long Beach, California, USA*, volume 97 of *Proceedings of Machine Learning Research*, pages 322–332. PMLR, 2019.

[3] Yikun Ban, Yuchen Yan, Arindam Banerjee, and Jingrui He. Ee-net: Exploitation-exploration neural networks in contextual bandits. In *The Tenth International Conference on Learning Representations, ICLR 2022, Virtual Event, April 25-29, 2022*. OpenReview.net, 2022.

[4] Peter L. Bartlett and Shahar Mendelson. Rademacher and gaussian complexities: Risk bounds and structural results. *J. Mach. Learn. Res.*, 3:463–482, 2002.

[5] Karsten M. Borgwardt and Hans-Peter Kriegel. Shortest-path kernels on graphs. In *Proceedings of the 5th IEEE International Conference on Data Mining (ICDM 2005), 27-30 November 2005, Houston, Texas, USA*, pages 74–81. IEEE Computer Society, 2005.

[6] Hong Chen and Hisashi Koga. Gl2vec: Graph embedding enriched by line graphs with edge features. In Tom Gedeon, Kok Wai Wong, and Minho Lee, editors, *Neural Information Processing - 26th International Conference, ICONIP 2019, Sydney, NSW, Australia, December 12-15, 2019, Proceedings, Part III*, volume 11955 of *Lecture Notes in Computer Science*, pages 3–14. Springer, 2019.

[7] Andrea Cini, Ivan Marisca, Filippo Maria Bianchi, and Cesare Alippi. Scalable spatiotemporal graph neural networks. In Brian Williams, Yiling Chen, and Jennifer Neville, editors, *Thirty-Seventh AAAI Conference on Artificial Intelligence, AAAI 2023, Thirty-Fifth Conference on Innovative Applications of Artificial Intelligence, IAAI 2023, Thirteenth Symposium on Educational Advances in Artificial Intelligence, EAAI 2023, Washington, DC, USA, February 7-14, 2023*, pages 7218–7226. AAAI Press, 2023.

[8] Weilin Cong, Si Zhang, Jian Kang, Baichuan Yuan, Hao Wu, Xin Zhou, Hanghang Tong, and Mehrdad Mahdavi. Do we really need complicated model architectures for temporal networks? In *The Eleventh International Conference on Learning Representations, ICLR 2023, Kigali, Rwanda, May 1-5, 2023*. OpenReview.net, 2023.

[9] Simon S. Du, Kangcheng Hou, Ruslan Salakhutdinov, Barnabás Póczos, Ruosong Wang, and Keyulu Xu. Graph neural tangent kernel: Fusing graph neural networks with graph kernels. In Hanna M. Wallach, Hugo Larochelle, Alina Beygelzimer, Florence d'Alché-Buc, Emily B. Fox, and Roman Garnett, editors, *Advances in Neural Information Processing Systems 32: Annual Conference on Neural Information Processing Systems 2019, NeurIPS 2019, December 8-14, 2019, Vancouver, BC, Canada*, pages 5724–5734, 2019.

[10] Simon S. Du, Jason D. Lee, Haochuan Li, Liwei Wang, and Xiyu Zhai. Gradient descent finds global minima of deep neural networks. In Kamalika Chaudhuri and Ruslan Salakhutdinov, editors, *Proceedings of the 36th International Conference on Machine Learning, ICML 2019, 9-15 June 2019, Long Beach, California, USA*, volume 97 of *Proceedings of Machine Learning Research*, pages 1675–1685. PMLR, 2019.

[11] Simon S. Du, Xiyu Zhai, Barnabás Póczos, and Aarti Singh. Gradient descent provably optimizes over-parameterized neural networks. In *7th International Conference on Learning Representations, ICLR 2019, New Orleans, LA, USA, May 6-9, 2019*. OpenReview.net, 2019.

[12] Dongqi Fu, Liri Fang, Ross Maciejewski, Vetle I. Torvik, and Jingrui He. Meta-learned metrics over multi-evolution temporal graphs. In Aidong Zhang and Huzefa Rangwala, editors, *KDD '22: The 28th ACM SIGKDD Conference on Knowledge Discovery and Data Mining, Washington, DC, USA, August 14 - 18, 2022*, pages 367–377. ACM, 2022.

[13] Dongqi Fu, Zhigang Hua, Yan Xie, Jin Fang, Si Zhang, Kaan Sancak, Hao Wu, Andrey Malevich, Jingrui He, and Bo Long. Vcr-graphormer: A mini-batch graph transformer via virtual connections. *CoRR*, abs/2403.16030, 2024.

[14] Dongqi Fu, Dawei Zhou, Ross Maciejewski, Arie Croitoru, Marcus Boyd, and Jingrui He. Fairness-aware clique-preserving spectral clustering of temporal graphs. In *Proceedings of the ACM Web Conference 2023, WWW 2023, Austin, TX, USA, 30 April 2023 - 4 May 2023*, pages 3755–3765. ACM, 2023.

[15] Dongqi Fu, Yada Zhu, Hanghang Tong, Kommy Weldemariam, Onkar Bhardwaj, and Jingrui He. Generating fine-grained causality in climate time series data for forecasting and anomaly detection. *CoRR*, abs/2408.04254, 2024.

[16] Xiaoxin He, Bryan Hooi, Thomas Laurent, Adam Perold, Yann LeCun, and Xavier Bresson. A generalization of vit/mlp-mixer to graphs. In Andreas Krause, Emma Brunskill, Kyunghyun Cho, Barbara Engelhardt, Sivan Sabato, and Jonathan Scarlett, editors, *International Conference on Machine Learning, ICML 2023, 23-29 July 2023, Honolulu, Hawaii, USA*, volume 202 of *Proceedings of Machine Learning Research*, pages 12724–12745. PMLR, 2023.

[17] Shenyang Huang, Farimah Poursafaei, Jacob Danovitch, Matthias Fey, Weihua Hu, Emanuele Rossi, Jure Leskovec, Michael Bronstein, Guillaume Rabusseau, and Reihaneh Rabbany. Temporal graph benchmark for machine learning on temporal graphs. *Advances in Neural Information Processing Systems*, 2023.

[18] Wei Huang, Yayong Li, Weitao Du, Richard Y. D. Xu, Jie Yin, Ling Chen, and Miao Zhang. Towards deepening graph neural networks: A gntk-based optimization perspective. In *The Tenth International Conference on Learning Representations, ICLR 2022, Virtual Event, April 25-29, 2022*. OpenReview.net, 2022.

[19] Arthur Jacot, Clément Hongler, and Franck Gabriel. Neural tangent kernel: Convergence and generalization in neural networks. In Samy Bengio, Hanna M. Wallach, Hugo Larochelle, Kristen Grauman, Nicolò Cesa-Bianchi, and Roman Garnett, editors, *Advances in Neural Information Processing Systems 31: Annual Conference on Neural Information Processing Systems 2018, NeurIPS 2018, December 3-8, 2018, Montréal, Canada*, pages 8580–8589, 2018.

[20] Shunhua Jiang, Yunze Man, Zhao Song, Zheng Yu, and Danyang Zhuo. Fast graph neural tangent kernel via kronecker sketching. In *Thirty-Sixth AAAI Conference on Artificial Intelligence, AAAI 2022, Thirty-Fourth Conference on Innovative Applications of Artificial Intelligence, IAAI 2022, The Twelveth Symposium on Educational Advances in Artificial Intelligence, EAAI 2022 Virtual Event, February 22 - March 1, 2022*, pages 7033–7041. AAAI Press, 2022.

[21] Wei Ju, Siyu Yi, Yifan Wang, Zhiping Xiao, Zhengyang Mao, Hourun Li, Yiyang Gu, Yifang Qin, Nan Yin, Senzhang Wang, Xinwang Liu, Xiao Luo, Philip S. Yu, and Ming Zhang. A survey of graph neural networks in real world: Imbalance, noise, privacy and OOD challenges. *CoRR*, abs/2403.04468, 2024.

[22] Seyed Mehran Kazemi, Rishab Goel, Kshitij Jain, Ivan Kobyzev, Akshay Sethi, Peter Forsyth, and Pascal Poupart. Representation learning for dynamic graphs: A survey. *J. Mach. Learn. Res.*, 21:70:1–70:73, 2020.

[23] Nils M. Kriege, Fredrik D. Johansson, and Christopher Morris. A survey on graph kernels. *Appl. Netw. Sci.*, 5(1):6, 2020.

[24] Sanjukta Krishnagopal and Luana Ruiz. Graph neural tangent kernel: Convergence on large graphs. In Andreas Krause, Emma Brunskill, Kyunghyun Cho, Barbara Engelhardt, Sivan Sabato, and Jonathan Scarlett, editors, *International Conference on Machine Learning, ICML 2023, 23-29 July 2023, Honolulu, Hawaii, USA*, volume 202 of *Proceedings of Machine Learning Research*, pages 17827–17841. PMLR, 2023.

[25] Srijan Kumar, Xikun Zhang, and Jure Leskovec. Predicting dynamic embedding trajectory in temporal interaction networks. In Ankur Teredesai, Vipin Kumar, Ying Li, Rómer Rosales, Evimaria Terzi, and George Karypis, editors, *Proceedings of the 25th ACM SIGKDD International Conference on Knowledge Discovery & Data Mining, KDD 2019, Anchorage, AK, USA, August 4-8, 2019*, pages 1269–1278. ACM, 2019.

[26] Vitaly Kuznetsov and Mehryar Mohri. Time series prediction and online learning. In Vitaly Feldman, Alexander Rakhlin, and Ohad Shamir, editors, *Proceedings of the 29th Conference on Learning Theory, COLT 2016, New York, USA, June 23-26, 2016*, volume 49 of *JMLR Workshop and Conference Proceedings*, pages 1190–1213. JMLR.org, 2016.

[27] Zihao Li, Dongqi Fu, and Jingrui He. Everything evolves in personalized pagerank. In Ying Ding, Jie Tang, Juan F. Sequeda, Lora Aroyo, Carlos Castillo, and Geert-Jan Houben, editors, *Proceedings of the ACM Web Conference 2023, WWW 2023, Austin, TX, USA, 30 April 2023 - 4 May 2023*, pages 3342–3352. ACM, 2023.

[28] Xiao Lin, Jian Kang, Weilin Cong, and Hanghang Tong. Bemap: Balanced message passing for fair graph neural network. In *Learning on Graphs Conference*, pages 37–1. PMLR, 2024.

[29] Xiao Lin, Zhining Liu, Dongqi Fu, Ruizhong Qiu, and Hanghang Tong. Backtime: Backdoor attacks on multivariate time series forecasting. *arXiv preprint arXiv:2410.02195*, 2024.

[30] Antonio Longa, Veronica Lachi, Gabriele Santin, Monica Bianchini, Bruno Lepri, Pietro Lio, Franco Scarselli, and Andrea Passerini. Graph neural networks for temporal graphs: State of the art, open challenges, and opportunities. *Trans. Mach. Learn. Res.*, 2023, 2023.

[31] Yao Ma, Ziyi Guo, Zhaochun Ren, Jiliang Tang, and Dawei Yin. Streaming graph neural networks. In Jimmy X. Huang, Yi Chang, Xueqi Cheng, Jaap Kamps, Vanessa Murdock, Ji-Rong Wen, and Yiqun Liu, editors, *Proceedings of the 43rd International ACM SIGIR conference on research and development in Information Retrieval, SIGIR 2020, Virtual Event, China, July 25-30, 2020*, pages 719–728. ACM, 2020.

[32] Christopher Morris, Nils M. Kriege, Franka Bause, Kristian Kersting, Petra Mutzel, and Marion Neumann. Tudataset: A collection of benchmark datasets for learning with graphs. *CoRR*, abs/2007.08663, 2020.

[33] Annamalai Narayanan, Mahinthan Chandramohan, Rajasekar Venkatesan, Lihui Chen, Yang Liu, and Shantanu Jaiswal. graph2vec: Learning distributed representations of graphs. *CoRR*, abs/1707.05005, 2017.

[34] Giannis Nikolentzos, Giannis Siglidis, and Michalis Vazirgiannis. Graph kernels: A survey. *J. Artif. Intell. Res.*, 72:943–1027, 2021.

[35] Aldo Pareja, Giacomo Domeniconi, Jie Chen, Tengfei Ma, Toyotaro Suzumura, Hiroki Kanezashi, Tim Kaler, Tao B. Schardl, and Charles E. Leiserson. Evolvegcn: Evolving graph convolutional networks for dynamic graphs. In *The Thirty-Fourth AAAI Conference on Artificial Intelligence, AAAI 2020, The Thirty-Second Innovative Applications of Artificial Intelligence Conference, IAAI 2020, The Tenth AAAI Symposium on Educational Advances in Artificial Intelligence, EAAI 2020, New York, NY, USA, February 7-12, 2020*, pages 5363–5370. AAAI Press, 2020.

[36] Yunzhe Qi, Yikun Ban, and Jingrui He. Neural bandit with arm group graph. In Aidong Zhang and Huzefa Rangwala, editors, *KDD '22: The 28th ACM SIGKDD Conference on Knowledge Discovery and Data Mining, Washington, DC, USA, August 14 - 18, 2022*, pages 1379–1389. ACM, 2022.

[37] Yunzhe Qi, Yikun Ban, and Jingrui He. Graph neural bandits. In Ambuj K. Singh, Yizhou Sun, Leman Akoglu, Dimitrios Gunopulos, Xifeng Yan, Ravi Kumar, Fatma Ozcan, and Jieping Ye, editors, *Proceedings of the 29th ACM SIGKDD Conference on Knowledge Discovery and Data Mining, KDD 2023, Long Beach, CA, USA, August 6-10, 2023*, pages 1920–1931. ACM, 2023.

[38] Alexander Rakhlin, Karthik Sridharan, and Ambuj Tewari. Online learning via sequential complexities. *J. Mach. Learn. Res.*, 16:155–186, 2015.

[39] Emanuele Rossi, Ben Chamberlain, Fabrizio Frasca, Davide Eynard, Federico Monti, and Michael M. Bronstein. Temporal graph networks for deep learning on dynamic graphs. *CoRR*, abs/2006.10637, 2020.

[40] Benedek Rozemberczki, Oliver Kiss, and Rik Sarkar. Karate Club: An API Oriented Open-source Python Framework for Unsupervised Learning on Graphs. In *Proceedings of the 29th ACM International Conference on Information and Knowledge Management (CIKM '20)*, page 3125–3132. ACM, 2020.

[41] Luana Ruiz, Luiz F. O. Chamon, and Alejandro Ribeiro. Graphon neural networks and the transferability of graph neural networks. In Hugo Larochelle, Marc'Aurelio Ranzato, Raia Hadsell, Maria-Florina Balcan, and Hsuan-Tien Lin, editors, *Advances in Neural Information Processing Systems 33: Annual Conference on Neural Information Processing Systems 2020, NeurIPS 2020, December 6-12, 2020, virtual*, 2020.

[42] Nino Shervashidze, Pascal Schweitzer, Erik Jan van Leeuwen, Kurt Mehlhorn, and Karsten M. Borgwardt. Weisfeiler-lehman graph kernels. *J. Mach. Learn. Res.*, 12:2539–2561, 2011.

[43] Giannis Siglidis, Giannis Nikolentzos, Stratis Limnios, Christos Giatsidis, Konstantinos Skianis, and Michalis Vazirgiannis. Grakel: A graph kernel library in python. *Journal of Machine Learning Research*, 21(54):1–5, 2020.

[44] Rakshit Trivedi, Mehrdad Farajtabar, Prasenjeet Biswal, and Hongyuan Zha. Dyrep: Learning representations over dynamic graphs. In *7th International Conference on Learning Representations, ICLR 2019, New Orleans, LA, USA, May 6-9, 2019*. OpenReview.net, 2019.

[45] Anton Tsitsulin, Davide Mottin, Panagiotis Karras, Alexander M. Bronstein, and Emmanuel Müller. Netlsd: Hearing the shape of a graph. In Yike Guo and Faisal Farooq, editors, *Proceedings of the 24th ACM SIGKDD International Conference on Knowledge Discovery & Data Mining, KDD 2018, London, UK, August 19-23, 2018*, pages 2347–2356. ACM, 2018.

[46] S. V. N. Vishwanathan, Karsten M. Borgwardt, Imre Risi Kondor, and Nicol N. Schraudolph. Graph kernels. *CoRR*, abs/0807.0093, 2008.

[47] Da Xu, Chuanwei Ruan, Evren Körpeoglu, Sushant Kumar, and Kannan Achan. Inductive representation learning on temporal graphs. In *8th International Conference on Learning Representations, ICLR 2020, Addis Ababa, Ethiopia, April 26-30, 2020*. OpenReview.net, 2020.

[48] Jiaxuan You, Tianyu Du, and Jure Leskovec. ROLAND: graph learning framework for dynamic graphs. In Aidong Zhang and Huzefa Rangwala, editors, *KDD '22: The 28th ACM SIGKDD Conference on Knowledge Discovery and Data Mining, Washington, DC, USA, August 14 - 18, 2022*, pages 2358–2366. ACM, 2022.

[49] Le Yu, Leilei Sun, Bowen Du, and Weifeng Lv. Towards better dynamic graph learning: New architecture and unified library. *CoRR*, abs/2303.13047, 2023.

[50] Dawei Zhou, Lecheng Zheng, Dongqi Fu, Jiawei Han, and Jingrui He. Mentorgnn: Deriving curriculum for pre-training gnns. In Mohammad Al Hasan and Li Xiong, editors, *Proceedings of the 31st ACM International Conference on Information & Knowledge Management, Atlanta, GA, USA, October 17-21, 2022*, pages 2721–2731. ACM, 2022.

[51] Hongkuan Zhou, Da Zheng, Israt Nisa, Vasileios Ioannidis, Xiang Song, and George Karypis. TGL: A general framework for temporal GNN training on billion-scale graphs. *CoRR*, abs/2203.14883, 2022.

# A Pseudo Code of Temp-G³NTK

The pseudo-code of Temp-G³NTK is provided below.

---

**Algorithm 1** Pseudo-code for computing Temp-G³NTK value between $G, G'$ at time $t$

---

**Input:** node embeddings of $G, G'$ at time $t$: $\mathbf{h}(t), \mathbf{h}'(t)$; number of BLOCK operations $L$
**Output:** Temp-G³NTK value between $G, G'$ at time $t$: $K$

1: $K = 0$
2: **for** $u \in V^{(t)}$ **do**
3:     **for** $u' \in V'^{(t)}$ **do**
4:         $\mathbf{\Theta}_{u,u'}^{(0)} \leftarrow \mathbf{h}_u(t)^T \mathbf{h}_{u'}(t)$;
5:         $\mathbf{\Sigma}_{u,u'}^{(0)} \leftarrow \mathbf{h}_u(t)^T \mathbf{h}_{u'}(t)$;
6:         **for** $l \in [1, \ldots, L]$ **do**
7:             $\mathbf{\Sigma}_{u,u'}^{(l)} \leftarrow \dfrac{\sqrt{1-\arccos\left(\mathbf{\Sigma}_{u,u'}^{(l-1)}\right)^2}}{2\pi}$
8:             $\dot{\mathbf{\Sigma}}_{u,u'}^{(l)} \leftarrow \dfrac{\pi-\arccos\left(\mathbf{\Sigma}_{u,u'}^{(l-1)}\right)}{2\pi}$
9:             $\mathbf{\Theta}_{u,u'}^{(l)} \leftarrow \mathbf{\Theta}_{u,u'}^{(l-1)} \cdot \dot{\mathbf{\Sigma}}_{u,u'}^{(l-1)} + \mathbf{\Sigma}_{u,u'}^{(l-1)}$
10:         **end for**
11:     **end for**
12: **end for**
13: **for** $u \in V^{(t)}$ **do**
14:     **for** $u' \in V'^{(t)}$ **do**
15:         $K \leftarrow K + \mathbf{\Theta}_{u,u'}^{(L)}$
16:     **end for**
17: **end for**

---

# B Theoretical proof for Kernel Properties of Temp-G³NTK

Here, we present the full proof for Theorem 5.1 and Theorem 5.2.

*Proof.* For Theorem 5.1, we aim to prove that $K(G^{(t)}, G'^{(t)}) = K(G'^{(t)}, G^{(t)})$.

Given our proposed kernel function, $K(G^{(t)}, G'^{(t)}) = \sum_{v \in V(t)} \sum_{v' \in V'(t)} \mathbf{\Theta}^{(L)}(G^{(t)}, G'^{(t)})_{vv'}$, we first write down another equation, where the internal order is flipped, i.e., $K(G'^{(t)}, G^{(t)}) = \sum_{v' \in V'(t)} \sum_{v \in V(t)} \mathbf{\Theta}^{(L)}(G'^{(t)}, G^{(t)})_{v'v}$

We first prove that

$$\mathbf{\Theta}^{(l)}(G^{(t)}, G'^{(t)})_{vv'} = \mathbf{\Theta}^{(l)}(G'^{(t)}, G^{(t)})_{v'v}, \forall l, 1 \leq l \leq L.$$

▷ For $l = 1$, we have

$$\mathbf{\Theta}^{(1)}(G^{(t)}, G'^{(t)})_{vv'} = \mathbf{\Theta}^{(0)}(G^{(t)}, G'^{(t)})_{vv'} \cdot \dot{\mathbf{\Sigma}}^{(1)}(G^{(t)}, G'^{(t)})_{vv'} + \mathbf{\Sigma}^{(1)}(G^{(t)}, G'^{(t)})_{vv'}$$

$$= (h_v(t)^\top h_{v'}(t)) \cdot \frac{\pi - \arccos(\mathbf{\Sigma}^{(0)}(G^{(t)}, G'^{(t)})_{vv'})}{2\pi}$$

$$+ \frac{\pi - \arccos(\mathbf{\Sigma}^{(0)}(G^{(t)}, G'^{(t)})_{vv'})}{2\pi} + \frac{\sqrt{1 - \mathbf{\Sigma}^{(0)}(G^{(t)}, G'^{(t)})_{vv'}^2}}{2\pi}$$

$$= (h_v(t)^\top h_{v'}(t)) \cdot \frac{\pi - \arccos(h_v(t)^\top h_{v'}(t))}{2\pi} + \frac{\pi - \arccos(h_v(t)^\top h_{v'}(t))}{2\pi} + \frac{\sqrt{1 - (h_v(t)^\top h_{v'}(t))^2}}{2\pi}$$

$$= (h_{v'}(t)^\top h_v(t)) \cdot \frac{\pi - \arccos(h_{v'}(t)^\top h_v(t))}{2\pi} + \frac{\pi - \arccos(h_{v'}(t)^\top h_v(t))}{2\pi} + \frac{\sqrt{1 - (h_{v'}(t)^\top h_v(t))^2}}{2\pi}$$

$$= (h_{v'}(t)^\top h_v(t)) \cdot \frac{\pi - \arccos(\mathbf{\Sigma}^{(0)}(G'^{(t)}, G^{(t)})_{v'v})}{2\pi}$$

$$+ \frac{\pi - \arccos(\mathbf{\Sigma}^{(0)}(G'^{(t)}, G^{(t)})_{v'v})}{2\pi} + \frac{\sqrt{1 - \mathbf{\Sigma}^{(0)}(G'^{(t)}, G^{(t)})_{v'v}^2}}{2\pi}$$

$$= \mathbf{\Theta}^{(0)}(G'^{(t)}, G^{(t)})_{v'v} \cdot \dot{\mathbf{\Sigma}}^{(1)}(G'^{(t)}, G^{(t)})_{v'v} + \mathbf{\Sigma}^{(1)}(G'^{(t)}, G^{(t)})_{v'v}$$

$$= \mathbf{\Theta}^{(1)}(G'^{(t)}, G^{(t)})_{v'v}$$

▷ Suppose $\exists k \in \mathbb{N}, 1 \leq k \leq L$, such that

$$\mathbf{\Theta}^{(k)}(G^{(t)}, G'^{(t)})_{vv'} = \mathbf{\Theta}^{(k)}(G'^{(t)}, G^{(t)})_{v'v}$$

Thus,

$$\mathbf{\Theta}^{(k+1)}(G^{(t)}, G'^{(t)})_{vv'} = \mathbf{\Theta}^{(k)}(G^{(t)}, G'^{(t)})_{vv'} \cdot \dot{\mathbf{\Sigma}}^{(k+1)}(G^{(t)}, G'^{(t)})_{vv'} + \mathbf{\Sigma}^{(k+1)}(G^{(t)}, G'^{(t)})_{vv'}$$

$$= \mathbf{\Theta}^{(k)}(G^{(t)}, G'^{(t)})_{vv'} \cdot \frac{\pi - \arccos(\mathbf{\Sigma}^{(k)}(G^{(t)}, G'^{(t)})_{vv'})}{2\pi}$$

$$+ \frac{\pi - \arccos(\mathbf{\Sigma}^{(k)}(G^{(t)}, G'^{(t)})_{vv'})}{2\pi} + \frac{\sqrt{1 - \mathbf{\Sigma}^{(k)}(G^{(t)}, G'^{(t)})_{vv'}^2}}{2\pi}$$

$$= \mathbf{\Theta}^{(k)}(G'^{(t)}, G^{(t)})_{v'v} \cdot \frac{\pi - \arccos(\mathbf{\Sigma}^{(k)}(G'^{(t)}, G^{(t)})_{v'v})}{2\pi}$$

$$+ \frac{\pi - \arccos(\mathbf{\Sigma}^{(k)}(G'^{(t)}, G^{(t)})_{v'v})}{2\pi} + \frac{\sqrt{1 - \mathbf{\Sigma}^{(k)}(G'^{(t)}, G^{(t)})_{v'v}^2}}{2\pi}$$

$$= \mathbf{\Theta}^{(k+1)}(G'^{(t)}, G^{(t)})_{v'v}$$

Therefore, if $\mathbf{\Theta}^{(k)}(G^{(t)}, G'^{(t)})_{vv'} = \mathbf{\Theta}^{(k)}(G'^{(t)}, G^{(t)})_{v'v}$, then $\mathbf{\Theta}^{(k+1)}(G^{(t)}, G'^{(t)})_{vv'} = \mathbf{\Theta}^{(k+1)}(G'^{(t)}, G^{(t)})_{v'v}$. Moreover, we have proven that $\mathbf{\Theta}^{(1)}(G^{(t)}, G'^{(t)})_{vv'} = \mathbf{\Theta}^{(1)}(G'^{(t)}, G^{(t)})_{v'v}$. Thus, by induction, we have:

$$\mathbf{\Theta}^{(l)}(G^{(t)}, G'^{(t)})_{vv'} = \mathbf{\Theta}^{(l)}(G'^{(t)}, G^{(t)})_{v'v}, \forall l, 1 \leq l \leq L.$$

Finally,

$$K(G(t), G'(t)) = \sum_{v \in V(t)} \sum_{v' \in V'(t)} \mathbf{\Theta}^{(L)}(G^{(t)}, G'^{(t)})_{vv'} = \sum_{v' \in V'(t)} \sum_{v \in V(t)} \mathbf{\Theta}^{(L)}(G'^{(t)}, G^{(t)})_{v'v} = K(G'(t), G(t))$$

The proof for Theorem 5.1 is completed. □

Next, we elaborate on the proof for Theorem 5.2. To be specific, we prepared two options to demonstrate the proof.

▷ Option #1:

*Proof.* In order to prove that Temp-G$^3$NTK is positive semi-definite, we need to prove the following statement. Given $n$ temporal graphs $G_1, \ldots, G_n$ and any $c_1, \ldots, c_n \in \mathbb{R}$ then

$$\sum_{i=1}^{n} \sum_{j=1}^{n} c_i c_j K(G_i^{(t)}, G_j^{(t)}) \geq 0 \tag{18}$$

Intuitively, we can view the right-hand side of Eq. 18 as the summation of all entries of the following matrix,

$$\mathbf{K} = \begin{pmatrix} c_1 c_1 K(G_1^{(t)}, G_1^{(t)}) & c_1 c_2 K(G_1^{(t)}, G_2^{(t)}) & \ldots & c_1 c_n K(G_1^{(t)}, G_n^{(t)}) \\ c_2 c_1 K(G_2^{(t)}, G_1^{(t)}) & c_2 c_2 K(G_2^{(t)}, G_2^{(t)}) & \ldots & c_2 c_n K(G_2^{(t)}, G_n^{(t)}) \\ \vdots & \vdots & \ddots & \vdots \\ c_n c_1 K(G_n^{(t)}, G_1^{(t)}) & c_n c_2 K(G_n^{(t)}, G_2^{(t)}) & \ldots & c_n c_n K(G_n^{(t)}, G_n^{(t)}) \end{pmatrix} \in \mathbb{R}^{n \times n} \tag{19}$$

whose $(i, j)^{\text{th}}$ entry is $c_i c_j K(G_i^{(t)}, G_j^{(t)})$.

Then, we can re-write Eq. 18 by Temp-G$^3$NTK's formula stated in Eq. 12 as follows

$$\sum_{i=1}^{n} \sum_{j=1}^{n} c_i c_j K(G_i^{(t)}, G_j^{(t)}) = \sum_{i=1}^{n} \sum_{j=1}^{n} \sum_{v=1}^{m_i} \sum_{v'=1}^{m_j} c_i c_j \mathbf{\Theta}^{(L)}(G_i^{(t)}, G_j^{(t)})_{vv'} \tag{20}$$

where $m_i$ is the number of nodes of $G_i$, $\forall i \in \{1, \ldots, n\}$.

Next, we consider the graph $G_1^{(t)} \cup G_2^{(t)} \cup \cdots \cup G_n^{(t)}$, whose vertex, edge set is the union of all $G_i^{(t)}$'s vertex, edge set ($\forall i \in \{1, \ldots, n\}$), respectively. Then the number of nodes of $G_1^{(t)} \cup G_2^{(t)} \cup \cdots \cup G_n^{(t)}$ if $\overline{m} = \sum_{i=1}^{n} m_i$. Additionally, we re-index the nodes of $G_1^{(t)} \cup G_2^{(t)} \cup \cdots \cup G_n^{(t)}$ as follows. The $j^{\text{th}}$ node of graph $G_i^{(t)}$ is the $((\sum_{p=1}^{i-1} m_p) + j)^{\text{th}}$ node of $G_1^{(t)} \cup G_2^{(t)} \cup \cdots \cup G_n^{(t)}$. Then,

$$\sum_{i=1}^{n} \sum_{j=1}^{n} \sum_{v=1}^{m_i} \sum_{v'=1}^{m_j} c_i c_j \mathbf{\Theta}^{(L)}(G_i^{(t)}, G_j^{(t)})_{vv'} =$$
$$\sum_{v=1}^{\overline{m}} \sum_{v'=1}^{\overline{m}} a_v a_v' \mathbf{\Theta}^{(L)}(G_1^{(t)} \cup G_2^{(t)} \cup \cdots \cup G_n^{(t)}, G_1^{(t)} \cup G_2^{(t)} \cup \cdots \cup G_n^{(t)})_{v,v'} \geq 0 \tag{21}$$

where $a_v = c_i$ if $\sum_{q=1}^{i-1} m_q < v \leq \sum_{p=1}^{i} m_p$ for $(i \geq 1)$, and if $v \leq m_1$ then $a_v = c_1$.

Intuitively, the right-hand side of Eq. 21 is the summation of all entries of the following matrix $\mathbf{O}$:

$$\mathbf{O} = \begin{pmatrix} c_1 c_1 \mathbf{\Theta}^{(L)}(G_1^{(t)}, G_1^{(t)}) & c_1 c_2 \mathbf{\Theta}^{(L)}(G_1^{(t)}, G_2^{(t)}) & \ldots & c_1 c_n \mathbf{\Theta}^{(L)}(G_1^{(t)}, G_n^{(t)}) \\ c_2 c_1 \mathbf{\Theta}^{(L)}(G_2^{(t)}, G_1^{(t)}) & c_2 c_2 \mathbf{\Theta}^{(L)}(G_2^{(t)}, G_2^{(t)}) & \ldots & c_2 c_n \mathbf{\Theta}^{(L)}(G_2^{(t)}, G_n^{(t)}) \\ \vdots & \vdots & \ddots & \vdots \\ c_n c_1 \mathbf{\Theta}^{(L)}(G_n^{(t)}, G_1^{(t)}) & c_n c_2 \mathbf{\Theta}^{(L)}(G_n^{(t)}, G_2^{(t)}) & \ldots & c_n c_n \mathbf{\Theta}^{(L)}(G_n^{(t)}, G_n^{(t)}) \end{pmatrix} \in \mathbb{R}^{\overline{m} \times \overline{m}} \tag{22}$$

whose $(i, j)^{\text{th}}$ entry is an $m_i \times m_j$ matrix $c_i c_j \mathbf{\Theta}^{(L)}(G_i^{(t)}, G_j^{(t)})$ and $\mathbf{\Theta}^{(L)}(G_i^{(t)}, G_j^{(t)})$ is the kernel matrix defined iteratively via Eq. 11 for all pair of nodes from $G_i^{(t)}$ and $G_j^{(t)}$. We can regard $\mathbf{O}$ (defined in Eq. 22) as the "node-view" expansion of $\mathbf{K}$ in Eq. 19.

As $a_1, \ldots, a_{\overline{m}} \in \mathbb{R}$, and $\mathbf{\Theta}^{(L)}$ is the kernel matrix constructed on feature vector of each pair of nodes of $G_1^{(t)} \cup \cdots \cup G_n^{(t)}$, so the last inequality in Eq. 21 holds, as $\mathbf{\Theta}^{(L)}$ is positive semi-definite on the space of node features vector [19].

Finally, we conclude that

$$\sum_{i=1}^{n}\sum_{j=1}^{n} c_i c_j K(G_i^{(t)}, G_j^{(t)}) =$$

$$\sum_{v=1}^{\overline{m}}\sum_{v'=1}^{\overline{m}} a_v a_v' \Theta^{(L)}(G_1^{(t)} \cup G_2^{(t)} \cup \cdots \cup G_n^{(t)}, G_1^{(t)} \cup G_2^{(t)} \cup \cdots \cup G_n^{(t)})_{v,v'} \geq 0 \qquad (23)$$

Therefore, the proof for Theorem 5.2 is completed. □

▷ Option #2:

*Proof.* In order to prove that Temp-G$^3$NTK is positive semi-definite, we need to prove the following statement. Given $n$ temporal graphs $G_1, \ldots, G_n$ and any $c_1, \ldots, c_n \in \mathbb{R}$ then

$$\sum_{i=1}^{n}\sum_{j=1}^{n} c_i c_j K(G_i^{(t)}, G_j^{(t)}) \geq 0 \qquad (24)$$

We re-write Eq. 24 as follows

$$\sum_{i=1}^{n}\sum_{j=1}^{n} c_i c_j K(G_i^{(t)}, G_j^{(t)}) =$$

$$= \left( \sum_{i=1}^{n}\sum_{j=1}^{n} c_i^2 K(G_i^{(t)}, G_i^{(t)}) + c_j^2 K(G_j^{(t)}, G_j^{(t)}) + c_i c_j K(G_i^{(t)}, G_j^{(t)}) + c_j c_i K(G_j^{(t)}, G_i^{(t)}) \right)$$

$$+ \sum_{i=1}^{n}\sum_{j=1}^{n} (-c_i^2)K(G_i, G_i) + (-c_j^2)K(G_j, G_j) + (-c_i c_j)K(G_i, G_j)$$

$$\Leftrightarrow 2\sum_{i=1}^{n}\sum_{j=1}^{n} c_i c_j K(G_i^{(t)}, G_j^{(t)}) = \sum_{i=1}^{n}\sum_{j=1}^{n} (-c_i^2)K(G_i, G_i) + (-c_j^2)K(G_j, G_j)$$

$$+ \left( \sum_{i=1}^{n}\sum_{j=1}^{n} c_i^2 K(G_i^{(t)}, G_i^{(t)}) + c_j^2 K(G_j^{(t)}, G_j^{(t)}) + c_i c_j K(G_i^{(t)}, G_j^{(t)}) + c_j c_i K(G_j^{(t)}, G_i^{(t)}) \right)$$

$$\Leftrightarrow 2\sum_{i=1}^{n}\sum_{j=1}^{n} c_i c_j K(G_i^{(t)}, G_j^{(t)}) = 2n \sum_{i=1}^{n} (-c_i^2)K(G_i, G_i)$$

$$+ \left( \sum_{i=1}^{n}\sum_{j=1}^{n} c_i^2 K(G_i^{(t)}, G_i^{(t)}) + c_j^2 K(G_j^{(t)}, G_j^{(t)}) + c_i c_j K(G_i^{(t)}, G_j^{(t)}) + c_j c_i K(G_j^{(t)}, G_i^{(t)}) \right)$$

Next, we aim to prove that (1) for each $i, j \in \{1, \ldots, n\}$,

$$\left( \sum_{i=1}^{n}\sum_{j=1}^{n} c_i^2 K(G_i^{(t)}, G_i^{(t)}) + c_j^2 K(G_j^{(t)}, G_j^{(t)}) + c_i c_j K(G_i^{(t)}, G_j^{(t)}) + c_j c_i K(G_j^{(t)}, G_i^{(t)}) \right) \geq 0 \quad (25)$$

by proving that for each $i, j \in \{1, \ldots, n\}$

$$c_i^2 K(G_i^{(t)}, G_i^{(t)}) + c_j^2 K(G_j^{(t)}, G_j^{(t)}) + c_i c_j K(G_i^{(t)}, G_j^{(t)}) + c_j c_i K(G_j^{(t)}, G_i^{(t)}) \geq 0$$

and (2) for each $i \in \{1, \ldots, n\}$,

$$2n \sum_{i=1}^{n} (-c_i^2) K(G_i^{(t)}, G_i^{(t)}) \geq 0 \tag{26}$$

by proving that for each $i \in \{1, \ldots, n\}$,

$$(-c_i^2) K(G_i^{(t)}, G_i^{(t)}) \geq 0$$

▷▷ *For proving (1):*

For each $i, j \in \{1, \ldots, n\}$, suppose that $G_i, G_j$ have $p, q$ nodes, respectively, then we have the following equality

$$c_i^2 K(G_i^{(t)}, G_i^{(t)}) + c_j^2 K(G_j^{(t)}, G_j^{(t)}) + c_i c_j K(G_i^{(t)}, G_j^{(t)}) + c_j c_i K(G_j^{(t)}, G_i^{(t)}) =$$

$$= c_i^2 \sum_{v=1}^{p} \sum_{v'=1}^{p} \Theta^{(L)}(G_i^{(t)}, G_i^{(t)})_{vv'} + c_j^2 \sum_{v=1}^{q} \sum_{v'=1}^{q} \Theta^{(L)}(G_j^{(t)}, G_j^{(t)})_{vv'} +$$

$$c_i c_j \sum_{v=1}^{p} \sum_{v'=1}^{q} \Theta^{(L)}(G_i^{(t)}, G_j^{(t)})_{vv'} + c_j c_i \sum_{v'=1}^{q} \sum_{v=1}^{p} \Theta^{(L)}(G_j^{(t)}, G_i^{(t)})_{v'v}$$

$$= \sum_{v=1}^{p+q} \sum_{v'=1}^{p+q} a_v a_v' \Theta^{(L)}(G_i^{(t)} \cup G_j^{(t)}, G_i^{(t)} \cup G_j^{(t)})_{vv'} \geq 0$$

Here, we can regard $G_i^{(t)} \cup G_j^{(t)}$ as a graph, whose vertex, edge set is the union of $G_i^{(t)}$'s vertex, edge set and $G_j^{(t)}$'s vertex, edge set, respectively, while $a_v, a_v'$ is either $c_i$ or $c_j$. Therefore, since $a_1, \ldots a_{p+q} \in \mathbb{R}$, and $\Theta^{(L)}(G_i^{(t)} \cup G_j^{(t)}, G_i^{(t)} \cup G_j^{(t)})_{vv'}, \forall v, v' \in \{1, \ldots, (p+q)\}$ is equivalent to constructing a kernel matrix on the $(p+q)$ node feature vectors of $G_i^{(t)} \cup G_j^{(t)}$, so the last inequality holds, as $\Theta^{(L)}$ is positive semi-definite on the space of node features vector [19]. Therefore, we can infer that

$$\left( \sum_{i=1}^{n} \sum_{j=1}^{n} c_i^2 K(G_i^{(t)}, G_i^{(t)}) + c_j^2 K(G_j^{(t)}, G_j^{(t)}) + c_i c_j K(G_i^{(t)}, G_j^{(t)}) + c_j c_i K(G_j^{(t)}, G_i^{(t)}) \right) \geq 0 \tag{27}$$

▷▷ *For proving (2):*

Next, for each $i$, suppose $G_i$ has $k$ nodes then

$$(-c_i^2) K(G_i^{(t)}, G_i^{(t)}) = (-c_i^2) \sum_{v=1}^{k} \sum_{v'=1}^{k} \Theta^{(L)}(G_i^{(t)}, G_i^{(t)})_{vv'} = \sum_{v=1}^{k} \sum_{v'=1}^{k} b_v b_{v'} \Theta^{(L)}(G_i^{(t)}, G_i^{(t)})_{vv'} \geq 0 \tag{28}$$

where $b_1 = \cdots = b_k = (-c_i^2)$.

As $b_1, \ldots, b_k \in \mathbb{R}$, and $\Theta^{(L)}(G_i^{(t)}, G_i^{(t)})_{vv'}, \forall v, v' \in \{1, \ldots, k\}$ is equivalent to the pair-wise kernel matrix constructed on the set of node features of $G_i$, so the inequality holds, due to the positive semi-definite on the node feature space property of $\Theta^{(L)}$ [19]. This results in

$$2n \sum_{i=1}^{n} (-c_i^2) K(G_i^{(t)}, G_i^{(t)}) \geq 0 \tag{29}$$

From Eq. 29 and Eq. 27, we now have

$$2\sum_{i=1}^{n}\sum_{j=1}^{n}c_ic_jK(G_i^{(t)},G_j^{(t)}) = 2n\sum_{i=1}^{n}(-c_i^2)K(G_i,G_i)$$

$$+ \left( \sum_{i=1}^{n}\sum_{j=1}^{n}c_i^2K(G_i^{(t)},G_i^{(t)}) + c_j^2K(G_j^{(t)},G_j^{(t)}) + c_ic_jK(G_i^{(t)},G_j^{(t)}) + c_jc_iK(G_j^{(t)},G_i^{(t)}) \right) \geq 0$$

$$\Rightarrow \sum_{i=1}^{n}\sum_{j=1}^{n}c_ic_jK(G_i^{(t)},G_j^{(t)}) \geq 0$$

Therefore, the proof for Theorem 5.2 is now completed.

$\square$

## C  Genralization Bound of Temp-G$^3$NTK

In this section, we provide the full proof for Theorem 5.3. We first provide some background knowledge on the Sequential Rademacher Complexity measures [38, 26], and then we derive at the full proof of Theorem 5.3.

### C.1  Preliminaries

In our Temporal Graph Classification setting, given a temporal $G$ and its label $y$, if we want to make predictions about $G$ at time $t$ then we apply our predictor on the snapshot $G^{(t)}$, i.e., we leverage all information at previous timestamps $\bar{t}$ ($\bar{t} < t$).

Suppose that $G$ has $T$ unique timestamps $t_1, \ldots, t_T$, then we can obtain $T$ snapshots $G_1, \ldots, G_T$, where $G_i = G^{(t_i)}$. Therefore, we can re-formulate our setting as follows: we consider a general time series prediction, where the predictor receives a realization $((G_1, t_1), \ldots, (G_T, t_T))$ generated by some stochastic processes.

To simplify notations, we let $f$ be the regression kernel predictor, i.e., $f_{kernel}$. The objective of our predictor $f$ is, at any timestamp $t_i$, achieving a small error $\mathbb{E}[\ell(f((G_i, t_i)), y)|((G_1, t_1), \ldots, (G_{i-1}, t_{i-1}))]$ conditioned on previous snapshots, given a loss function $\ell : \mathbb{R} \times \mathbb{R} \to \mathbb{R}$.

For shorter notation, we let $g(Z) = \ell(f((G_i, t_i), y)$ for $Z = ((G_i, t_i), y) \in \mathcal{Z}$ and let the family function $\mathcal{G} = \{((G_i, t_i), y) \to \ell(f((G_i, t_i), y)\}$ contain such functions $g$. We assume a bounded, $\alpha$−Lipschitz loss function, that is $g(Z) \in [0, 1]$ for any $Z \in \mathcal{Z}$. Finally, we use $\mathbf{Z}_a^b$ to denote the sequences $Z_a, Z_{a+1}, \ldots, Z_b$, where $Z_i = ((G_i, t_i), y)$.

In order to derive the Sequential Rademacher Complexity, we first introduce the definition of a complete binary tree.

We adopt the following definition of a complete binary tree from [38, 26]: a $\mathcal{Z}$−valued complete binary tree $\mathbf{z}$ is a sequence of $(z_1, \ldots, z_T)$ of $T$ mappings, where $z_i : \{\pm 1\}^{i-1} \to \mathcal{Z}$. A path in the tree is $\sigma = (\sigma_1, \ldots, \sigma_{T-1})$. To simplify the notation, we write $z_i(\boldsymbol{\sigma}) = (\sigma_1, \ldots, \sigma_{i-1})$.

Next, we introduce how to sample sequential data $Z_1, \ldots, Z_i$ using the aforementioned binary tree. We adopt the sampling process from [38, 26] as follows: given a stochastic process distributed to the distribution $\mathbb{P}$ with $\mathbb{P}_i(.|\mathbf{z}_1^{i-1})$, denoting the conditional distribution based on $z_1, \ldots, z_{i-1}$, we sample a $\mathcal{Z} \times \mathcal{Z}$ based on the following procedure. We start by drawing two independent samples $Z_1, Z_1'$ from $\mathbb{P}_1$, then, in the left child of the root we sample $Z_2, Z_2' \sim \mathbb{P}_2(.|Z_1)$ and in the right child of the root, we sample $Z_2, Z_2' \sim \mathbb{P}_2(.|Z_1')$.

More generally, for a node that can be reached by a path $(\sigma_1, \ldots, \sigma_{i-1})$, we draw $Z_i, Z_i' \sim \mathbb{P}_i(.|I_1(\sigma_1), \ldots, I_{i-1}(\sigma_{i-1}))$, where the indicator $I_j(1) = Z_j, I_j(-1) = Z_j'$. In this manner, we derive at the Sequential Rademacher Complexity of a function class $\mathcal{G}$ that acts on $\mathbf{z}$ is defined as follows [38]:

$$\mathfrak{R}_T^{\text{seq}}(\mathcal{G}, \mathbf{z}) = \mathbb{E}\left[ \sup_{g \in \mathcal{G}} \frac{1}{T} \sum_{i=1}^{T} g(z_i(\boldsymbol{\sigma})) \right] \tag{30}$$

where $\mathbf{z}$ is an $\mathcal{Z}-$valued complete binary trees with depth $T$ and $\boldsymbol{\sigma}$ is a sequence of Rademacher random variables.

As stated in Theorem 5.3, the key quantity of interest in our analysis is

$$\sup_{\ell \in \mathcal{L}} \left[ \frac{1}{T} \sum_{i=1}^{T} \mathbb{E}[\ell(f((G_i, t_i)), y)|((G_1, , t_1), \dots, (G_{i-1}, t_{i-1}))] - \ell(f((G_i, t_i)), y) \right]$$

and we can re-write this quantity as follows and establish a data-dependent bound for this term in Appendix C.2

$$\sup_{g \in \mathcal{G}} \left[ \frac{1}{T} \sum_{i=1}^{T} \mathbb{E}[g(Z_i)|\mathbf{Z}_1^{(i-1)}] - g(Z_i) \right] \tag{31}$$

For more details about the Sequential Complexity measure, we defer readers to [38] and [26].

## C.2  Detailed Proofs

We first bound Eq. 31 by the Sequential Rademacher complexity of $\mathcal{F}$, the family function class contains functions such as our kernel regression predictor, $f_{kernel}$, (Lemma C.1) then continue to bound the Sequential Rademacher complexity of $\mathcal{F}$ by the data-dependent term (Lemma. C.2).

**Lemma C.1.**

$$\sup_{g \in \mathcal{G}} \left[ \frac{1}{T} \sum_{i=1}^{T} \mathbb{E}[g(Z_i)|\mathbf{Z}_1^{(i-1)}] - g(Z_i) \right] \leq 2\alpha \mathfrak{R}_T^{seq}(\mathcal{F}) \tag{32}$$

*Proof.* We first state that the following inequalities hold: $\mathbb{E}[g(Z_i)|\mathbf{Z}_1^{i-1}] = \mathbb{E}[g(Z_i')|\mathbf{Z}_1^{i-1}]$, since $Z_i, Z_i'$ are indepedently drawn from $\mathbb{P}_i(.|\mathbf{Z}_1^{i-1}))$ and $\mathbb{E}[g(Z_i)|\mathbf{Z}_1^{i-1}] = \mathbb{E}[g(Z_i)|\mathbf{Z}_1^{T}]$, and $g(Z_i)$ only depends on $\mathbf{Z}_1^{i-1}$. Therefore, we obtain the following:

$$\mathbb{E}\left[ \sup_{g \in \mathcal{G}} \left[ \frac{1}{T} \sum_{i=1}^{T} \mathbb{E}[g(Z_i)|\mathbf{Z}_1^{(i-1)}] - g(Z_i) \right] \right]$$

$$= \mathbb{E}\left[ \sup_{g \in \mathcal{G}} \frac{1}{T} \mathbb{E}\left[ \sum_{i=1}^{T} (g(Z_i') - g(Z_i))|\mathbf{Z}_1^{T} \right] \right]$$

$$\leq \mathbb{E}\left[ \frac{1}{T} \mathbb{E}\left[ \sup_{g \in \mathcal{G}} \sum_{i=1}^{T} (g(Z_i') - g(Z_i)) \right] \right]$$

$$= \frac{1}{T} \mathbb{E}\left[ \sup_{g \in \mathcal{G}} \sum_{i=1}^{T} (g(Z_i') - g(Z_i)) \right]$$

where the first inequality holds by using Jensen's inequality, and the last expectation is taken over all joint sequences $\mathbf{Z}_1^{T}, \mathbf{Z'}_1^{T}$.

Since $g(Z_i) = \ell(f((G_i, t_i), y)$ and $\ell$ is $\alpha-$Lipschitz, thus we obtain the following:

$$\mathbb{E}\left[ \sup_{g \in G} \sum_{i=1}^{T} (g(Z_i') - g(Z_i)) \right]$$

$$= \mathbb{E}\left[ \sup_{\ell \in \mathcal{L}} \sum_{i=1}^{T} \ell(f((G_i', t_i')), y) - \ell(f((G_i, t_i)), y) \right]$$

$$\leq \mathbb{E}\left[ \sup_{f \in \mathcal{F}} \alpha \sum_{i=1}^{T} ((f((G_i', t_i')) - f((G_i, t_i)) + (y - y)) \right] \tag{33}$$

$$= \alpha \mathbb{E}\left[ \sup_{f \in \mathcal{F}} \sum_{i=1}^{T} ((f(X_i') - f(X_i)) \right]$$

The first inequality holds, due to the fact that $\ell$ is $\alpha-$Lipschitz.

where $X_i = (G_i, t_i), X_i' = (G_i', t_i')$. Note that since $y$ is fixed with respect to $i$ ($1 \le i \le T$), so $Z_i = (X_i, y)$. Therefore, we can derive at a $\mathcal{X}$-valued complete binary tree $\mathbf{x}$, $\mathbf{x}_i(\boldsymbol{\sigma})$, and the sequences $\mathbf{X}_1^T$, $\mathbf{X'}_1^T$ that are similar to the manner of deriving at $\mathbf{z}, \mathbf{z}_i(\boldsymbol{\sigma}), \mathbf{Z}_1^T, \mathbf{Z'}_1^T$, respectively. Simply say, $\mathbf{x}$ is $\mathbf{z}$ when omitting the label $y$. Since the last expectation is taken over all joint sequences $\mathbf{X}_1^T, \mathbf{X'}_1^T$, so given Rademacher random variables, $\sigma_1, \ldots, \sigma_T$,

$$\mathbb{E}\left[\sup_{f \in \mathcal{F}} \sum_{i=1}^T (f(X_i') - f(X_i))\right] = \mathbb{E}[\sup_{f \in \mathcal{F}} \sum_{i=1}^T \sigma_i(f(X_i') - f(X_i))]$$

Thus, we have:

$$\mathbb{E}\left[\sup_{f \in \mathcal{F}} \sum_{i=1}^T (f(X_i') - f(X_i))\right]$$

$$= \mathbb{E}\left[\sup_{f \in \mathcal{F}} \sum_{i=1}^T \sigma_i(f(X_i') - f(X_i))\right]$$

$$= \mathbb{E}_{X_1, X_1' \sim \mathbb{P}_1} \ldots \mathbb{E}_{X_T, X_T' \sim \mathbb{P}_i(.|I_1(\sigma_1), \ldots I_T(\sigma_T))}\left[\sup_{f \in \mathcal{F}} \sum_{i=1}^T \sigma_i(f(X_i') - f(X_i))\right]$$

$$= \mathbb{E}_{\boldsymbol{\sigma}} \mathbb{E}_{\mathbf{x} \sim T(\mathbb{P})}\left[\sup_{f \in \mathcal{F}} \sum_{i=1}^T \sigma_i(f(X_i') - f(X_i))\right]$$

$$= \mathbb{E}_{\boldsymbol{\sigma}} \mathbb{E}_{(\mathbf{x}, \mathbf{x}') \sim T(\mathbb{P})}\left[\sup_{f \in \mathcal{F}} \sum_{i=1}^T \sigma_i(f(\mathbf{x}_i'(\boldsymbol{\sigma})) - f(\mathbf{x}_i(\boldsymbol{\sigma})))\right]$$

where $\mathbf{x} \sim T(\mathbb{P})$ denotes sampling a $\mathcal{X}$-valued complete binary tree $\mathbf{x}$ with a given stochastic process $\mathbb{P}$. Thus, Eq. 33 is equivalent to:

$$\alpha\mathbb{E}\left[\sup_{f \in \mathcal{F}} \sum_{i=1}^T (f(X_i') - f(X_i))\right]$$

$$= \alpha\mathbb{E}_{\boldsymbol{\sigma}} \mathbb{E}_{(\mathbf{x}, \mathbf{x}') \sim T(\mathbb{P})}\left[\sup_{f \in \mathcal{F}} \sum_{i=1}^T \sigma_i(f(\mathbf{x}_i'(\boldsymbol{\sigma})) - f(\mathbf{x}_i(\boldsymbol{\sigma})))\right]$$

$$\le \alpha\mathbb{E}_{\boldsymbol{\sigma}} \mathbb{E}_{(\mathbf{x}, \mathbf{x}') \sim T(\mathbb{P})}\left[\sup_{f \in \mathcal{F}} \sum_{i=1}^T \sigma_i f(\mathbf{x}_i'(\boldsymbol{\sigma})) + \sup_{f \in \mathcal{F}} \sum_{i=1}^T -\sigma_i f(\mathbf{x}_i(\boldsymbol{\sigma}))\right]$$

$$= \alpha\mathbb{E}_{\boldsymbol{\sigma}} \mathbb{E}_{(\mathbf{x}, \mathbf{x}') \sim T(\mathbb{P})}\left[\sup_{f \in \mathcal{F}} \sum_{i=1}^T \sigma_i f(\mathbf{x}_i'(\boldsymbol{\sigma}))\right] + \mathbb{E}_{\boldsymbol{\sigma}} \mathbb{E}_{(\mathbf{x}, \mathbf{x}') \sim T(\mathbb{P})}\left[\sup_{f \in \mathcal{F}} \sum_{i=1}^T -\sigma_i f(\mathbf{x}_i(\boldsymbol{\sigma}))\right] \quad (34)$$

$$= 2\alpha\mathbb{E}_{\boldsymbol{\sigma}} \mathbb{E}_{\mathbf{x} \sim T(\mathbb{P})}\left[\sup_{f \in \mathcal{F}} \sum_{i=1}^T \sigma_i f(\mathbf{x}_i(\sigma))\right]$$

$$= 2\alpha\mathbb{E}_{\mathbf{x} \sim T(\mathbb{P})}\left[\mathfrak{R}_T^{\text{seq}}(\mathcal{F}, \mathbf{x})\right]$$

which completes the proof for Lemma C.1. $\square$

Next, we establish a data-dependent bound for the Sequential Complexity measure $\mathfrak{R}^{\text{seq}}$.

**Lemma C.2.** *Given $n$ i.i.d time series samples drawn from an underlying stochastic processes $\mathbb{P}$, $\{(X^{(j)}, y_j)\}_{j=1}^n$. Then*

$$\frac{1}{n}\sum_{j=1}^n \mathfrak{R}_T^{seq}(\mathcal{F}, \mathbf{x}^{(j)}) \le \frac{2}{n}\sum_{i=1}^T \sqrt{\mathbf{y}^T[\mathbf{K}^{(i)}]^{-1}\mathbf{y} \cdot \text{tr}(\mathbf{K}^{(i)})} \quad (35)$$

*where, $\mathbf{x}^{(j)}$ is the binary tree corresponding to the time series $X^{(j)}$, $\mathbf{K}^{(i)}$ is the $n \times n$ kernel gram matrix, whose pq-th entry is the Temp-G$^3$NTK value of the $i$−th snapshot of $X^{(p)}$ and the $i$-th snapshot of $X^{(q)}$, and $\mathbf{y}$ is the vector of labels, in which the $j$-th entry is $[\mathbf{y}]_j = y_j$.*

*Proof.*

$$\frac{1}{n}\sum_{j=1}^{n}\mathfrak{R}_T^{\text{seq}}(\mathcal{F},\mathbf{x}^{(j)}) = \frac{1}{n}\sum_{j=1}^{n}\sup_{f\in\mathcal{F}}\sum_{i=1}^{T}\sigma_{ji}f(\mathbf{x}_i^{(j)}(\boldsymbol{\sigma}_j))$$

$$= \sum_{i=1}^{T}\sup_{f\in\mathcal{F}}\frac{1}{n}\sum_{j=1}^{n}\sigma_{ji}f(X_i^{(j)}) = \sum_{i=1}^{T}\hat{\mathfrak{R}}_n(\mathcal{F},i)$$

where $\hat{\mathfrak{R}}_n(\mathcal{F},i)$ is the empirical Rademacher complexity of $\mathcal{F}$ with the $i$-th snapshot of $n$ i.i.d samples $X_i^{(1)},\ldots,X_i^{(n)}$. Since $\mathcal{F}$ is a function class of kernel regression function, as proven in [4], we can bound $\hat{\mathfrak{R}}_n(\mathcal{F},i)$ as follows:

$$\hat{\mathfrak{R}}_n(\mathcal{F},i) \leq \frac{2}{n}\sqrt{\mathbf{y}^T[\mathbf{K}^{(i)}]^{-1}\mathbf{y}\cdot\text{tr}(\mathbf{K}^{(i)})}$$

which completes the proof for Lemma C.2. □

Thus, using the results of Lemma C.2, we can bound Eq. 34 as follows:

$$2\alpha\mathbb{E}_{\mathbf{x}\sim T(\mathbb{P})}\left[\mathfrak{R}_T^{\text{seq}}(\mathcal{F},\mathbf{x})\right] = 2\alpha\mathbb{E}_{\mathbf{x}\sim T(\mathbb{P})}\left[\frac{1}{n}\sum_{j=1}^{n}\mathfrak{R}_T^{\text{seq}}(\mathcal{F},\mathbf{x}^{(j)})\right]$$

$$\leq 2\alpha\mathbb{E}\left[\frac{2}{n}\sum_{i=1}^{T}\sqrt{\mathbf{y}^T[\mathbf{K}^{(i)}]^{-1}\mathbf{y}\cdot\text{tr}(\mathbf{K}^{(i)})}\right]$$

$$\leq \alpha\sup_{i}\frac{4}{n}\sqrt{\mathbf{y}^T[\mathbf{K}^{(i)}]^{-1}\mathbf{y}\cdot\text{tr}(\mathbf{K}^{(i)})}$$

which completes the proof for Theorem 5.3.

# D  Convergence to Graphon Neural Tangent Kernel

In this section, we first provide some background knowledge on Graphons, Graphons Neural Networks, and Graphons Neural Tangent Kernel, and then derive at the full proof for Theorem. 5.4.

## D.1  Preliminaries

We adopt the definition of Graphons, Graphons Neural Networks (WNN) [24], and Graphons Neural Tangent Kernel (WNTK) [24], and then extend these concepts to the settings of CTDGs.

### Graphons

Graphons are defined as bounded, symmetric, measurable function $W : [0,1]^2 \to [0,1]$ representing limits of sequences of dense graphs.

Given a graph sequence $\{G_n\}$, where the $i$-th graph in the sequence $G_i$ has $i$ nodes, let $\mathbf{F} = (V', E')$ be an undirected graph, then the graph sequence $\{G_n\}$ is said to converge to the graphon $W$ in the sense that

$$\lim_{n\to\infty} t(\mathbf{F}, G_n) = t(\mathbf{F}, W) \tag{36}$$

where $t(\mathbf{F}, G_n) = \text{hom}(\mathbf{F}, G_n)/n^{|V'|}$, where $\text{hom}(\mathbf{F}, G_n)$ is the number of homomorphisms between $\mathbf{F}$ and $G_n$, and $t(\mathbf{F}, W)$ can be similarly defined.

Thus, $t(\mathbf{F}, G_n)$ is the density of homomorphisms between $F$ and $G_n$. We can think of $\mathbf{F}$ as motifs such as $k$−cycles, or $k$−cliques, so if the graph sequence $\{G_n\}$ converges to the graphon $W$, then

we can think of $G_1, \ldots, G_n$ of the graph sequence $\{G_n\}$ belongs to a graph family that has a certain amount of density of homomorphisms from $\mathbf{F}$, and that graph family is represented by $W$.

Therefore, the graphon $W$ can be seen as a generative model for stochastic graphs. In order to use $W$ to generate a graph $G_n = (V_n, E_n)$ with $n$ nodes, we first map each node $i \in V_n (1 \leq i \leq n)$ to the unit interval, i.e. $[0, 1]$, by uniformly sampling points $u_i (u_i \in [0, 1])$, and the probability of nodes $i, j$ are connected in $G_n$ is $W(u_i, u_j)$, hence we can regard $W$ is a weighted adjacency matrix for $G_n$.

**Graphons Neural Network**

Next, we would define a continuous message passing framework for graphon that corresponds to the neural architecture proposed in Section 3.1.

Firstly, we introduce the definition of Graphon signals as follows: Graphon signals are function $X : [0, 1]^2 \rightarrow \mathbb{R}$, and $X$ has finite energy, i.e., $X \in L_2([0, 1]^2)$. In this way, we can think of $X(u_i, u_j)$ as the edge representation of an edge $(i, j)$. We adopt the definition of graphon signals from [24] and extend it to edge features, instead of graphon signals function for node feature as stated in [24].

Analogous to the sum neighborhood aggregation operation in equation (1), the aggregation operation for graphon $W$ and graphon signal $X$ can be defined as the function $T_W X : [0, 1] \rightarrow \mathbb{R}^d$:

$$T_W X(u) = \int_0^1 W(u, v) X(u, v) dv \tag{37}$$

Let $h \equiv T_W X$. If the aggregated information is further transformed by $L$ layers of MLPs (similar to Eq. 2) then for $l \in [L]$, $h^{(l)}$ is determined as follows:

$$h^{(l)}(u) = \sigma(\mathbf{H}^{(l)} h^{(l-1)}) \tag{38}$$

where $h^{(0)} = h$, $\mathbf{H}$ is the linear transformation and $\sigma$ is the non-linear ReLU activation function.

**Induced Graphon Neural Network for CTDGs**

We adopt the definition of induced graphon and induced graphon signals from [24], extend them to the CTDGs setting, and determine an induced graphon neural network that correspond to our proposed temporal graph learning algorithm in Section 3.1.

Given a CTDG $G$ and the graphon $W$ that represents a graph family of $G$, we would leverage the aforementioned graphon $W$ to determine the induced graphon and induced graphon signals that correspond to snapshots of a CTDG. At time $t$, let the number of nodes of $G^{(t)}$ be $n(t)$, let $W^{(t)} : [0, 1]^2 \rightarrow [0, 1]$ and $X^{(t)} : [0, 1]^2 \rightarrow \mathbb{R}^d$ denotes the induced graphon and induced graphon signals correspond to $G^{(t)}$, respectively. We determine $W^{(t)}$ and $X^{(t)}$ as the followings:

For any $u, v \in [0, 1]$, let $I_i = \left[(i-1)/n(t), i/n(t)\right), 1 \leq i \leq n(t)$ and $\bar{I}_i = \left[(i-1)/n(\bar{t}), i/n(\bar{t})\right), 1 \leq i \leq n(\bar{t})$. If $u \in I_i, v \in I_j$, where $1 \leq i, j \leq n(t)$, then

$$W^{(t)}(u, v) = \frac{W^{(\bar{t})}(u_{\bar{i}}, u_{\bar{j}}) \mathbb{I}(i \leq n(\bar{t})) \mathbb{I}(j \leq n(\bar{t})) + A^{(t)}(u_i, u_j)}{2 \cdot \mathbb{I}(i \leq n(\bar{t})) \mathbb{I}(j \leq n(\bar{t}))} \tag{39}$$

where $\bar{t} < t$, $u_i = (i-1)/n(t)$, $u_{\bar{i}} = (i-1)/n(\bar{t})$, $\mathbb{I}$ is the indicator function, and $A^{(t)}(u_i, u_j) \sim Ber(W(u_i, u_j))$, where $Ber$ indicates the Bernoulli distribution. The initial state, i.e., $t = 0$ would simply be $W^{(0)} = A^{(0)}(u_i, u_j)$, and we define the temporal graphon signal function at time $t$ as:

$$X^{(t)}(u, v) = \int_0^t A^{(\bar{t})}(u_{\bar{i}}, u_{\bar{j}}) \mathbb{I}(i \leq n(\bar{t})) \mathbb{I}(j \leq n(\bar{t})) d\bar{t} \tag{40}$$

and we let the graphon signals function that associated with $W$ be $X(u, v) = W(u, v)$.

In a similar manner to Equation (1), the sum aggregation opertaion at time $t$ would be:

$$T_{W^{(t)}} X^{(t)}(u) = \int_0^1 W^{(t)}(u, v) X^{(t)}(u, v) dv \tag{41}$$

and the final result is $T_{W^{(t)}}X^{(t)}$ after $L$ layers of MLP transformation, similar as Eq. 2.

### Graphon NTK and Induced Graphon Temp-G$^3$NTK

Let $f_{wnn}$ be the WNN defined by Eq. 37, 38, and $f_{temp-wnn}$ be the induced graphon neural networks defined by Eq. 39, 41, and 40. Similar to Eq. 6, given 2 graphons $W, W'$ and their signals $X, X'$ that correspond to 2 CTDGs $G, G'$, respectively, also given parameters $\mathbf{H}$, then the NTK of $f_{wnn}(X, W, \mathbf{H})$ and $f_{wnn}(X', W', \mathbf{H})$ would be:

$$\mathbf{\Theta}_{wnn}(X, X', W, W') = \mathbb{E}_{\mathbf{H} \sim \mathcal{N}(0,1)} \left\langle \frac{\partial f_{wnn}(X, W, \mathbf{H})}{\partial \mathbf{H}}, \frac{\partial f_{wnn}(X', W', \mathbf{H})}{\partial \mathbf{H}} \right\rangle$$

and the Temp-G$^3$NTK of the induced graphon neural network at time $t$ would be:

$$\mathbf{\Theta}_{temp-wnn}(X^{(t)}, X'^{(t)}, W^{(t)}, W'^{(t)}) = \mathbb{E}_{\mathbf{H} \sim \mathcal{N}(0,1)} \left\langle \frac{\partial f_{temp-wnn}(X^{(t)}, W^{(t)}, \mathbf{H})}{\partial \mathbf{H}}, \frac{\partial f_{temp-wnn}(X'^{(t)}, W'^{(t)}, \mathbf{H})}{\partial \mathbf{H}} \right\rangle$$

### D.2  Detailed Proofs

We let $K_W(W, W') \equiv \mathbf{\Theta}_{wnn}(X, X', W, W')$ and $K_W(W^{(t)}, W'^{(t)}) \equiv \mathbf{\Theta}_{temp-wnn}(X^{(t)}, X'^{(t)}, W^{(t)}, W'^{(t)})$, as we derive at the graphon signals $X, X'$ by $W, W'$ and the induced graphon signals $X^{(t)}, X'^{(t)}$ by $W^{(t)}, W'^{(t)}$, so in Theorem Section 5.4, we decide to denote the NTK by $K_W(W, W')$ and the induced NTK by $K_W(W^{(t)}, W'^{(t)})$ for simpler notation. For simplicity, we let the number of BLOCK operations be $L = 1$.

*Proof.*

$||K_W(W, W') - K_W(W^{(t)}, W'^{(t)})|| =$

$||\mathbf{\Theta}_{wnn}(X, X', W, W', \mathbf{H}) - \mathbf{\Theta}_{temp-wnn}(X^{(t)}, X'^{(t)}, W^{(t)}, W'^{(t)}, \mathbf{H})|| =$

$= ||\sigma'(\mathbf{H}T_W X)T_W X \cdot \sigma'(\mathbf{H}T_{W'}X')T_{W'}X' - \sigma'(\mathbf{H}T_{W^{(t)}}X^{(t)})T_{W^{(t)}}X^{(t)} \cdot \sigma'(\mathbf{H}T_{W'^{(t)}}X'^{(t)})T_{W'^{(t)}}X'^{(t)}||$

$= ||\sigma'(\mathbf{H}T_W X)T_W X \cdot \sigma'(\mathbf{H}T_{W'}X')T_{W'}X' - \sigma'(\mathbf{H}T_{W^{(t)}}X^{(t)})T_{W^{(t)}}X^{(t)} \cdot \sigma'(\mathbf{H}T_{W'}X')T_{W'}X'$

$+ \sigma'(\mathbf{H}T_{W^{(t)}}X^{(t)})T_{W^{(t)}}X^{(t)} \cdot \sigma'(\mathbf{H}T_{W'}X')T_{W'}X' - \sigma'(\mathbf{H}T_{W^{(t)}}X^{(t)})T_{W^{(t)}}X^{(t)} \cdot \sigma'(\mathbf{H}T_{W'^{(t)}}X'^{(t)})T_{W'^{(t)}}X'^{(t)}||$

$\leq ||\sigma'(\mathbf{H}T_W X)T_W X \cdot \sigma'(\mathbf{H}T_{W'}X')T_{W'}X' - \sigma'(\mathbf{H}T_{W^{(t)}}X^{(t)})T_{W^{(t)}}X^{(t)} \cdot \sigma'(\mathbf{H}T_{W'}X')T_{W'}X'||$

$+ ||\sigma'(\mathbf{H}T_{W^{(t)}}X^{(t)})T_{W^{(t)}}X^{(t)} \cdot \sigma'(\mathbf{H}T_{W'}X')T_{W'}X' - \sigma'(\mathbf{H}T_{W^{(t)}}X^{(t)})T_{W^{(t)}}X^{(t)} \cdot \sigma'(\mathbf{H}T_{W'^{(t)}}X'^{(t)})T_{W'^{(t)}}X'^{(t)}||$

$= \left|\left|\left(\sigma'(\mathbf{H}T_W X)T_W X - \sigma'(\mathbf{H}T_{W^{(t)}}X^{(t)})T_{W^{(t)}}X^{(t)}\right) \cdot \sigma'(\mathbf{H}T_{W'}X')T_{W'}X'\right|\right|$

$+ \left|\left|\left(\sigma'(\mathbf{H}T_{W'}X')T_{W'}X' - \sigma'(\mathbf{H}T_{W'^{(t)}}X'^{(t)})T_{W'^{(t)}}X'^{(t)}\right) \cdot \sigma'(\mathbf{H}T_{W^{(t)}}X^{(t)})T_{W^{(t)}}X^{(t)}\right|\right|$

$\leq \left|\left|\left(\sigma'(\mathbf{H}T_W X)T_W X - \sigma'(\mathbf{H}T_{W^{(t)}}X^{(t)})T_{W^{(t)}}X^{(t)}\right)\right|\right| \cdot \left|\left|\sigma'(\mathbf{H}T_{W'}X')T_{W'}X'\right|\right|$

$+ \left|\left|\left(\sigma'(\mathbf{H}T_{W'}X')T_{W'}X' - \sigma'(\mathbf{H}T_{W'^{(t)}}X'^{(t)})T_{W'^{(t)}}X'^{(t)}\right)\right|\right| \cdot \left|\left|\sigma'(\mathbf{H}T_{W^{(t)}}X^{(t)})T_{W^{(t)}}X^{(t)}\right|\right|$

$$(42)$$

The first inequality holds due to the triangle inequality, and the second inequality holds due to the property of the operator norm.

We can see that, in order to prove Theorem 5.4, it is sufficient to prove that:

$$\left|\left|\left(\sigma'(\mathbf{H}T_W \mathbf{x})T_W X - \sigma'(\mathbf{H}T_{W^{(t)}}X^{(t)})T_{W^{(t)}}X^{(t)}\right)\right|\right| \to 0 \tag{43}$$

and

$$\left|\left|\left(\sigma'(\mathbf{H}T_{W'}X')T_{W'}X' - \sigma'(\mathbf{H}T_{W'^{(t)}}X'^{(t)})T_{W'^{(t)}}X'^{(t)}\right)\right|\right| \to 0 \tag{44}$$

Since Eq. 43 and Eq. 44 are essentially the same, we would focus on proving Eq. 43. We further applying algebraic manipulation on Eq. 43 as follows:

$$
\begin{aligned}
&\left\|\sigma'(\mathbf{H}T_W X)T_W X - \sigma'(\mathbf{H}T_{W^{(t)}}X^{(t)})T_{W^{(t)}}X^{(t)}\right\| \\
&= \left\|\sigma'(\mathbf{H}T_W \mathbf{x})T_W X - \sigma'(\mathbf{H}T_W X)T_{W^{(t)}}X^{(t)} + \sigma'(\mathbf{H}T_W X)T_{W^{(t)}}X^{(t)} - \sigma'(\mathbf{H}T_{W^{(t)}}X^{(t)})T_{W^{(t)}}X^{(t)}\right\| \\
&\leq \left\|\sigma'(\mathbf{H}T_W X)T_W X - \sigma'(\mathbf{H}T_W X)T_{W^{(t)}}X^{(t)}\right\| + \left\|\sigma'(\mathbf{H}T_W X)T_{W^{(t)}}X^{(t)} - \sigma'(\mathbf{H}T_{W^{(t)}}X^{(t)})T_{W^{(t)}}X^{(t)}\right\| \\
&\leq \left\|\sigma'(\mathbf{H}T_W X)\right\| \cdot \left\|T_W X - T_{W^{(t)}}X^{(t)}\right\| + \left\|\sigma'(\mathbf{H}T_W X) - \sigma'(\mathbf{H}T_{W^{(t)}}X^{(t)})\right\| \cdot \left\|T_{W^{(t)}}X^{(t)}\right\|
\end{aligned}
$$

(45)

The first inequality holds due the triangle inequality, and the second inequality holds due to the property of the operator norm.

Therefore, from here, in order to prove Eq. 43, it is sufficient to prove that

$$
\left\|T_W X - T_{W^{(t)}}X^{(t)}\right\| \to 0 \tag{46}
$$

and

$$
\left\|\sigma'(\mathbf{H}T_W X) - \sigma'(\mathbf{H}T_{W^{(t)}}X^{(t)})\right\| \to 0 \tag{47}
$$

Since Eq .46 implies Eq. 47, so we would focus on proving Eq. 46. We further transform Eq. 46 as follows:

$$
\begin{aligned}
&\left\|T_W X - T_{W^{(t)}}X^{(t)}\right\| \\
&= \left\|T_W X - T_W X^{(t)} + T_W X^{(t)} - T_{W^{(t)}}X^{(t)}\right\| \\
&\leq \left\|T_W X - T_W X^{(t)}\right\| + \left\|T_W X^{(t)} - T_{W^{(t)}}X^{(t)}\right\| \\
&\leq \|T_W\| \cdot \|X - X^{(t)}\| + \|T_W - T_{W^{(t)}}\| \cdot \|X^{(t)}\|
\end{aligned}
$$

(48)

Similar to Eq. 45, we first apply the triangle inequality to obtain the first inequality, and apply the property of the operator norm to determine the second inequailty.

From here, it is sufficient if prove that

$$
\|X - X^{(t)}\| \to 0 \tag{49}
$$

and

$$
\|T_W - T_{W^{(t)}}\| \to 0 \tag{50}
$$

We observe that

$$
\|X - X^{(t)}\| \leq \|W - W^{(t)}\| \tag{51}
$$

the inequality holds due to the definition of $X^{(t)}$, thus by Lemma D.1, Eq. 49 holds.

As proven in [24], if both $\|X - X^{(t)}\|$ and $\|W - W^{(t)}\|$ converges to 0 as $t \to \infty$, then Eq. 50 holds.

Therefore, the proof for Theorem 5.4 is now completed.

$\square$

**Lemma D.1.** $||W - W^{(t)}||$ *is bounded by* $1/n(t)^2$ *and* $1/n(\bar{t})^2$ *and thus,*

$$||W - W^{(t)}|| \to 0, \text{ as } t \to \infty \tag{52}$$

*since* $1/n(t)^2 \to 0$ *as* $t \to \infty$

*Proof.* In order to prove the convergence of this lemma, we will need to prove the convergence in $L_2$ norm, i.e.,

$$||W - W^{(t)}||_{L_2} \to 0$$

We first prove that

$$||W - A^{(t)}||_{L_2} \leq \frac{4\beta}{n(t)^2} \tag{53}$$

$$
\begin{aligned}
||W - A^{(t)}||_{L_2} &= \int_0^1 \int_0^1 (W(u,v) - A^{(t)}(u,v))^2 du\,dv = \\
&= \sum_{i=1}^{n(t)} \sum_{j=1}^{n(t)} \int_{(i-1)/n(t)}^{i/n(t)} \int_{(j-1)/n(t)}^{j/n(t)} (W(u,v) - A^{(t)}(u,v))^2 du\,dv \\
&= \sum_{i=1}^{n(t)} \sum_{j=1}^{n(t)} \int_{(i-1)/n(t)}^{i/n(t)} \int_{(j-1)/n(t)}^{j/n(t)} (W(u,v) - W(u_i, u_j))^2 du\,dv \\
&\leq \beta \sum_{i=1}^{n(t)} \sum_{j=1}^{n(t)} \int_{(i-1)/n(t)}^{i/n(t)} \int_{(j-1)/n(t)}^{j/n(t)} (|u - u_i| + |v - u_j|)^2 du\,dv \\
&\leq \beta \sum_{i=1}^{n(t)} \sum_{j=1}^{n(t)} \int_{(i-1)/n(t)}^{i/n(t)} \int_{(j-1)/n(t)}^{j/n(t)} \frac{4}{n(t)^2} du\,dv \\
&= \beta \sum_{i=1}^{n(t)} \sum_{j=1}^{n(t)} \frac{4}{n(t)^4} \\
&= \beta n(t)^2 \cdot \frac{4}{n(t)^4} = \beta \frac{4}{n(t)^2}
\end{aligned}
\tag{54}
$$

The first inequality holds due the fact that $W, W'$ are $\beta-$Lipschitz.

Next, we prove the lemma using mathematical induction.

For $t = 0$, then $W^{(0)} = A^{(0)}$, thus the lemma holds for $W^{(0)}$, since $||W - W^{(0)}|| = ||W - A^{(0)}||$ is bounded by $\frac{4\beta}{n(0)^2}$. Therefore, the Lemma holds for $t = 0$.

Given that we fix some timestamp $\bar{t}$, and suppose that the Lemma holds for every timestamp in the interval $[0, \bar{t}]$. We would focus on proving that the Lemma also holds for $t > \bar{t}$

$$
\begin{aligned}
||W - W^{(t)}||_{L_2} &= \\
&= \frac{1}{2}||W - W^{(\bar{t})} + W - A^{(t)}|| \\
&\leq \frac{1}{2}||W - W^{(\bar{t})}|| + \frac{1}{2}||W - A^{(t)}|| \\
&\leq \frac{1}{2}||W - W^{(t)}|| + \frac{2\beta}{n(t)^2}
\end{aligned}
\tag{55}
$$

The first term is bounded by $1/n(\bar{t})^2$ by our induction hypothesis, and the second term is also bounded by $1/n(t)^2$, and as $t \to \infty, \bar{t} \to \infty$ then $1/n(t)^2, 1/n(\bar{t})^2 \to 0$ and thus $||W - W^{(t)}||_{L_2} \to 0$ as $t \to \infty$, which completes the proof. $\qquad \square$

### D.3 Detailed Comparison with Previous Work

In short, we break through the convergence limitations of previous work [24]. For instance, besides temporal dependencies between snapshots of the evolving graphs, the work [24] that is most closely related to our theoretical results does not account for any dependencies between static graphs and does not establish a limit object for two different graphs. The detailed illustration is delivered as follows.

- We first summarize the setting about the convergence of static graphs to graphons and the theoretical findings of [24]: the previous work that is most closely to our theoretical findings in Theorem 5.4 is [24]. [24] proves that if a graph sequence of static random graphs $\{G_n\}$ with growing number of nodes, i.e., the number of nodes in $G_i$ is less than or equal $G_j$ with $i < j$, converges to a graphon $\mathbf{W}$, and the graph signal sequences $\{x_n\}$, $\{x'_n\}$ converge to the graphon signal $X, X'$, respectively, then the induced graphon neural tangent kernel between graph $G_i$ with signal $x_i$ and graph $G_i$ with signal $x'_i$ converges to the graphon neural tangent kernel between $W$ with signal $X$ and $W$ with signal $X'$, as the number of nodes in the graphs of the sequence $\{G_n\}$ goes to infinity.

- Therefore, we notice that the theoretical findings in [24] do not account for any dependencies between static graphs in the graph sequence, and [24] establishes the results of convergence for the neural tangent kernel between the same graph, but with different signals.

- Next, we point out how evolving graphs can be represented as a sequence of graphs with a growing number of nodes. Suppose we have an evolving graphs $G$ that has $n$ snapshots, $G^{(t_1)}, \ldots, G^{(t_n)}$ and the number of nodes in $G^{(t_i)}$ is less than or equal the number of nodes in $G^{(t_j)}$ if $i < j$. Then the snapshots of $G$ can be regarded as a graph sequence with the growing number of nodes. However, unlike the graph sequence in [24], there are temporal dependencies between graphs in the graph sequence of $G$.

- Similar to the result of [24] for static graphs, we establish a limit object on the graph sequence representation of the evolving graph, and overcome the limitations of [24], as we take the temporal dependencies between graphs in the graph sequence of the evolving graph into account, and derive at the limit object for the graphon neural tangent kernel of 2 different evolving graphs.

## E  Detailed Comparison with Previous Temporal Graph Representation Learning Works

In this section, we provide detailed comparison between our Temp-G$^3$NTK and previous traditional temporal graph learning methods, DGNN [31], EvolveGCN [35], ROLAND [48], and SSGNN [7], that rely on neural network training (e.g, stacking neural layers, gradient descent, and backpropagation) to obtain neural representations to support corresponding graph downstream tasks. However, our Temp-G$^3$NTK does not rely on neural network structure but can achieve the expressive power of graph neural networks, as our theoretical analysis and experiments demonstrate.

To be specific, DGNN [31], EvolveGCN [35], ROLAND [48], and SSGNN [7] belong to the category of recurrent graph neural architectures that handle temporal information (e.g., on the learnable weight level like EvolveGCN [31] or hidden representation level like ROLAND [48]). This is indeed an effective direction, but it requires heavy time complexity.

Facing this problem, an emerging direction appears, i.e., MLP-Mixer on Static Graphs [16] or GraphMixer on temporal graphs [8]. Especially, GraphMixer aggregates information from recent temporal neighbors and processes them with MLP-Mixer layers. Motivated by this direction, we propose our temporal graph neural tangent kernel. Also, we would like to note that, even without recurrent neural architectures, temporal information can also be preserved in our method.

To be more specific, in our method, temporal dependencies are captured in Eq. 1, where we construct the temporal node representations for a node at time by aggregating information (e.g., node features, edge features, and time difference) from its previous temporal neighbors. And the entire process does not involve neural training but just depends on mathematical time kernel functions. In other words, this process records the current state of based on its own neighborhood at the previous time

and can be retrieved for future state computation. Besides theoretical derivation, especially, Table 2 and Figure 1 visualize our method's effectiveness.

To support our above statement, we list the detailed illustration below.

- DGNN [31] is designed to execute the link prediction task (while ours is mainly for temporal graph classification and can be easily adapted to temporal node classification). In order to obtain the link predictions, nodes are categorized as interacting or influenced as follows. If two nodes are involved in a (directed) interaction $(u, v, t)$, then $u, v$ are interacting nodes and nodes that are nearby this interaction are referred to as "influenced nodes". On the other hand, If two nodes $u$ and $v$ interact at a certain time, then DGNN [31] updates their temporal representation. First, process each of $u$ and $v$ separately by employing recurrent architecture to their previous temporal representations and finally combine with time encoding the difference between the current time and the last interaction time of that node. Then, merge the two representations and obtain two new representations for $u$ and $v$. If two nodes interact, nearby nodes ("influenced nodes") would be affected. DGNN [31] also updates "influenced nodes", i.e., applying recurrent architecture on their previous representation, combining with two representations from interacting nodes, and the time encoding of difference between current time and last interacting time.

- EvolveGCN [35] is a link prediction and node classification method. Specifically, EvolveGCN [35] operates on graph snapshots and uses recurrent architecture (e.g., LSTM) to update the weight of each neural layer across time. Then, at a certain time $t$, EvolveGCN [35] gets node representation by applying the current snapshot's adjacency matrix, learnable weights, and representation from the previous recurrent layer.

- ROLAND [48] is also a method designed for link prediction. Different from EvolveGCN [31], the the recurrent architectures in ROLAND [48] are added on the hidden representation vectors other than learnable weights across timestamps. Then the components in recurrent architectures get simplified in ROLAND [48], which can be mainly based on MLPs and simple GNNs.

- SSGNN [7] is performs the time series forecasting task. To be more specific, SSGNN [7] first uses a deep randomized recurrent neural network to encode the history of each node encodings into high-dimensional vector embeddings, and then uses powers of the graph adjacency matrix to build informative node representations of the spatiotemporal dynamics at different scales. Then, the decoder maps the node representations into the desired output, e.g., future values of the time series.

Next, we state the position of our method in temporal graph learning. A recent temporal graph learning survey [30] reviewed temporal graph learning methods, including EvolveGCN [35], SSGNN [7], and DGNN [31], in their taxonomy, as plotted in its Figure 2 [30].

In that taxonomy, according to the best of our knowledge, our method belongs to the category "Event-based", and is the child node of "Temporal Embedding" and "Temporal Neighborhood", the position is close to the work TGL [51].

However, different from TGL [51], a large-scale graph neural network, to our best knowledge, our method is the first temporal graph neural tangent kernel method.

# F  Extra Temporal Graph-Level Experiments

## F.1  Ablation Study of Temporal Graph Classification

We conduct an ablation study to investigate how different time encoding functions affect the performance of Temp-G$^3$NTK. We select the infectious dataset to perform our ablation study.

Here, we examine how the usage of the time encoding function and its variations affect the performance of our predictor. Recall that we leverage $\mathbf{t}_{enc}$ to encode the raw relative difference between timestamps, $(t - \bar{t})$ in Eq. 1. For our ablation study, instead of the relative time difference encoding, i.e., $\mathbf{t}_{enc}(t - \bar{t})$, we also consider the raw relative difference, $(t - \bar{t})$, the absolute time encoding, $\mathbf{t}_{enc}(\bar{t})$, and the absolute timestamp, $\bar{t}$. The results are presented in Table 5, and the best accuracy is highlighted in bold. As we can see, the utilization of the time encoding function (first and second

Table 5: Ablation Study of Different Time Encoding Functions on Classification Accuracy on the INFECTIOUS Dataset.

| METHOD | ACCURACY |
|---|---|
| ABSOLUTE DIFFERENCE | $0.600 \pm 0.114$ |
| ABSOLUTE DIFFERENCE ENCODING | $0.570 \pm 0.081$ |
| RELATIVE DIFFERENCE | $0.620 \pm 0.060$ |
| TEMP-G$^3$NTK (RELATIVE DIFFERENCE ENCODING) | $\mathbf{0.740 \pm 0.058}$ |

Table 6: Parameter Analysis of Different Number of Recent Neighbors on Classification Accuracy on the INFECTIOUS Dataset.

| NUMBER OF RECENT NEIGHBORS | ACCURACY |
|---|---|
| 5 NEIGHBORS | $0.690 \pm 0.115$ |
| 10 NEIGHBORS | $0.700 \pm 0.126$ |
| 15 NEIGHBORS | $0.730 \pm 0.103$ |
| 20 NEIGHBORS | $0.710 \pm 0.097$ |
| 25 NEIGHBORS | $0.720 \pm 0.103$ |
| TEMP-G$^3$NTK (ALL NEIGHBORS) | $\mathbf{0.740 \pm 0.058}$ |

rows) yields higher accuracy than using raw timestamps, with $74\%$ and $62\%$ improvement, respectively. This suggests the ability and efficiency of $\mathbf{t}_{enc}$ in distinguishing different timestamps, which enhances the classification result.

## F.2 Parameter Analysis of Temporal Graph Classification

We conduct parameter analysis to investigate how different numbers of neighbors affect the performance of Temp-G$^3$NTK. We also select the infectious dataset to perform our parameter analysis.

Thus, we delve into how the Temp-G$^3$NTK performs with respect to the number of temporal neighbors. Specifically, in practice, most temporal graph representation learning methods aggregate information from the most $K$ recent neighbors [39, 8], instead of the full temporal neighborhood, $\mathcal{N}^{(t)}(v)$. For our parameter analysis, we vary the number of recent neighbors from $\{5, 10, 15, 20, 25\}$, perform the neighborhood aggregation (Eq. 1) on these recent neighbors, and report the classification accuracy of Temp-G$^3$NTK for the infectious dataset. The results are shown in Table 6, and the best results are highlighted in bold. Integrating all temporal neighbors into node representation yields higher accuracy than accounting for some recent neighbors, as shown in Table 6. These findings further suggest Temp-G$^3$NTK is able to leverage and capture the information in the full temporal neighborhood.

## F.3 Temporal Graph Similarity

Here, we use four unlabeled large real-world temporal graphs, WIKI, REDDIT, MOOC, and LASTFM to demonstrate the scalability of our Temp-G$^3$NTK, as shown in Figure 2 and Figure 3, where $x$-axis is the timestamp, and the $y$-axis is the similarity between two temporal graphs at a certain timestamp.

For each pair of temporal graphs, we compute the neural tangent kernel values with respect to time and plot the values as below. For each plot, the $y$-axis represents the Temp-G$^3$NTK value and the $x$-axis represents the timestamp. Although the timestamp ranges for each temporal graph are different, we rescale the $x$-axis to [0; 1000] for a better illustration. For each pair of graphs, the corresponding plot shows a different curve, suggesting that our Temp-G$^3$NTK can distinguish different pairs of temporal graphs. More interestingly, the corresponding observations align with our theoretical assumptions that the similarity of different growing temporal graphs tends to converge.

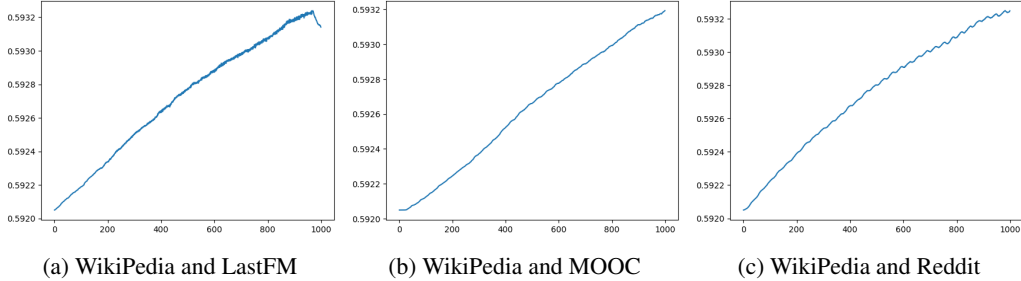

(a) WikiPedia and LastFM      (b) WikiPedia and MOOC      (c) WikiPedia and Reddit

Figure 2: Similarity of Different Temporal Graphs With Time Increased (Part I). $y$-axis represents the Temp-G$^3$NTK value, and the $x$-axis represents the timestamp

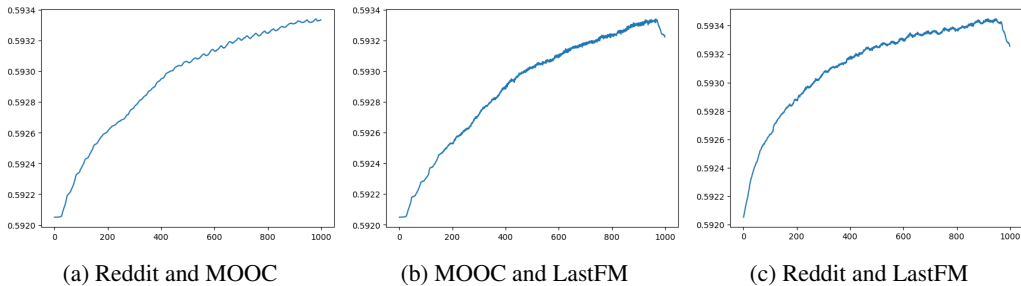

(a) Reddit and MOOC      (b) MOOC and LastFM      (c) Reddit and LastFM

Figure 3: Similarity of Different Temporal Graphs With Time Increased (Part II). $y$-axis represents the Temp-G$^3$NTK value, and the $x$-axis represents the timestamp

# G  Reproducibility

## G.1  Datasets Details

The detailed statistics of small and large temporal graph datasets for graph-level experiments are shown in Table 7 and Table 8.

Table 7: Small Temporal Graph Dataset Statistics

| DATASET | # GRAPHS | # CLASSES | # AVG NODES | # AVG EDGES |
| --- | --- | --- | --- | --- |
| INFECTIOUS | 200 | 2 | 50.00 | 459.72 |
| DBLP | 755 | 2 | 52.87 | 99.78 |
| FACEBOOK | 995 | 2 | 95.72 | 101.72 |
| TUMBLR | 373 | 2 | 53.11 | 71.63 |

## G.2  Temporal Graph Classification

Next, we provide details on how we conduct our experiments and the implementations of baseline algorithms in Section 6.1. In general, upon obtaining the time representation as in Eq. 1, we let the dimension of the time representation be $d_t = 25$ and $\alpha = \beta = \sqrt{d_t}$.

In order to leverage Temp-G$^3$NTK for graph classification, we employ C-SVM as a kernel regression predictor with the gram matrix of pairwise Temp-G$^3$NTK values of the training set as the pre-computed kernel. The regularization parameter $C$ of the SVM classifier is sampled evenly from 120 values in the interval $[10^{-2}, 10^4]$, in log scale, and set the number of maximum iterations to $5 \cdot 10^5$. For the number of BLOCK operations in our Temp-G$^3$NTK formula, $L$, we search for $L$ over $\{1, 2, 3\}$, and we notice that the validation accuracy remains unchanged while the $L$ varies.

For Graph Kernels and Graph Representation Learning methods, we first obtain the representation of each graph in the training set and then compute the pair-wise gram matrix, where each entry is the dot product of the representation of a graph pair. We then perform graph classification by

Table 8: Large Temporal Graph Dataset Statistics

| DATASET | # USERS | # ITEMS | # INTERACTIONS |
|---|---|---|---|
| REDDIT | 10,000 | 984 | 672,447 |
| WIKIPEDIA | 8,227 | 1,000 | 157,474 |
| LASTFM | 980 | 1,000 | 1,293,103 |
| MOOC | 7,047 | 97 | 411,749 |

leveraging C-SVM as our predictor and set the pre-compputed kernel as the aforementioned gram matrix. For the classifier regularization parameter $C$, we also determine this value by even sampling over the interval $[10^{-2}, 10^4]$, in log scale, and let the number of iterations be $5 \cdot 10^5$. We adopt the implementations of Graph Kernels from GRAKEL library [43] and the implementations Graph Representation Learning methods from the Karate Club library [40]. We adopt the default hyper-parameters from implementations of both libraries.

For TGL methods, which are TGN and GraphMixer, we first obtain node representations in each graph of the training set using the official code released by authors of TGN[8] and GraphMixer[9], then we determine the graph representation by performing sum pooling over the node representations. In the process of obtaining node representations, we adopt default hyper-parameters in the code of TGN and GraphMixer. Finally, we implement a simple linear classifier that consists of 1 layer of linear transformation and ReLU activation function, and the final output is determined by the Sigmoid function, as all datasets are binary classification. We train and optimize these models by the Adam optimizer, with the learning rate of $0.001$, $(\beta_1, \beta_2) = (0.9, 0.999)$, and the Binary Cross Entropy loss function. We adopt all default hyper-parameters.

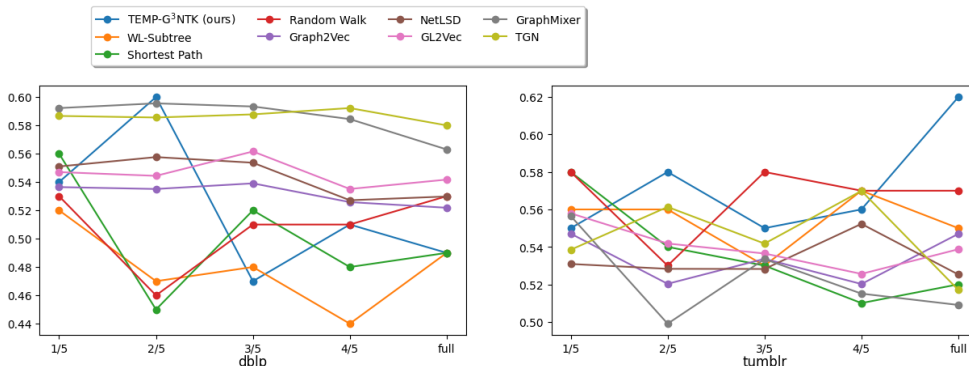

Figure 4: Plots of testing classification accuracy score of baseline algorithms with respect to different stages of temporal graphs from the dblp and tumblr datasets. The $y$-axis in each plot states the accuracy score, and the values in the $x$-axis represent how many percentages of timestamps have been taken into account. For example, at $x = 1/5$, the score is obtained by performing classification on the first $1/5$ timestamps of each graph.

In addition, we also provide the plot that illustrates the performance of baseline algorithms at different timestamps of the DBLP and TUMBLR datasets in Figure 4.

### G.3  Temporal Node Property Prediction

In order to utilize Temp-G$^3$NTK for node property prediction, we first compute the node pair-wise gram matrix at the last MLP layer in the Temp-G$^3$NTK formula, i.e the kernel matrix $\Theta^{(L)}$ in (9). Similar to performing graph classification task, we perform kernel regression with C-SVM and employ $\Theta^{(L)}$ as the pre-computed kernel, and $C$ is also searched over $[10^{-2}, 10^4]$ in log scale. We

also vary the number of BLOCK Operations, $L$, by $\{1, 2, 3\}$ to find the best NCDG score, which is obtained by $L = 1$.

